# Alias-Free Mamba Neural Operator

Jianwei Zheng, Wei Li, Ni Xu, Junwei Zhu, Xiaoxu Lin, and Xiaoqin Zhang*

Zhejiang University of Technology, Hangzhou, Zhejiang

## Abstract

Benefiting from the booming deep learning techniques, neural operators (NO) are considered as an ideal alternative to break the traditions of solving Partial Differential Equations (PDE) with expensive cost. Yet with the remarkable progress, current solutions concern little on the holistic function features–both global and local information– during the process of solving PDEs. Besides, a meticulously designed kernel integration to meet desirable performance often suffers from a severe computational burden, such as GNO with $O(N(N-1))$, FNO with $O(NlogN)$, and Transformer-based NO with $O(N^2)$. To counteract the dilemma, we propose a mamba neural operator with $O(N)$ computational complexity, namely MambaNO. Functionally, MambaNO achieves a clever balance between global integration, facilitated by state space model of Mamba that scans the entire function, and local integration, engaged with an alias-free architecture. We prove a property of continuous-discrete equivalence to show the capability of MambaNO in approximating operators arising from universal PDEs to desired accuracy. MambaNOs are evaluated on a diverse set of benchmarks with possibly multi-scale solutions and set new state-of-the-art scores, yet with fewer parameters and better efficiency.

## 1 Introduction

Numerous scientific and engineering problems entail recurrently resolving intricate Partial Differential Equation (PDE) [1] for various parameter values, including morphology across biomes [2], physical system state [3], and Radcliffe wave oscillation [4], to name just a few. Nonetheless, traditional solvers, such as Finite Element Methods (FEM) and Finite Difference Methods (FDM), necessitate an equilibrium between the speed of obtaining solutions and the level of refinement due to the requirement of resolving equations through domain discretization. Another disadvantage of these methods lies in the unavoidable resolving stage when the initial conditions of PDEs are varied. Recently, data-driven methods have emerged as an alternative for the development of faster, more robust, and more accurate solvers due to the natural ability to directly ascertain the trajectory of an PDE family from the data, instead of a single instance of the equation.

These techniques, collectively termed operator learning or neural operator, strive to approximate a well-behaved mapping from input function spaces, such as initial and boundary conditions, to solutions of some PDEs valued also within function spaces. As a burgeoning subject, most well-known NOs have been substantiated, encompassing Operator networks [5], DeepONets (DON) [6], Graph Neural Operator (GNO) [7], Fourier Neural Operator (FNO) [8], [9, 10], Transformer-based learning architectures [11, 12], and the most recent Convolution Neural Operators (CNO) [13].

Despite the substantial success of recently introduced operator learning frameworks, existing algorithms continue to demonstrate specific limitations to various degrees. The focus of operator network learning is on mapping between finite-dimensional function spaces, yet the truthful NO often engages with functions owning infinite dimensions. The practice of DON cannot take inputs at any point,

---

while GNO delivers very low efficiency when dealing with heterogeneous data. Transformer-based architectures suffer from slow inference speeds due to the quadratic complexity of attention computation, which serves as a special kernel integration. The use of Galerkin-style attention [14] can only alleviate rather than solve this problem. FNO accelerates the efficiency of the model by converting global convolution operation, which is a special form of kernel integration, into frequency domain multiplication through Fourier transformation. However, FNO may not respect the framework of alias-free operator learning, mentioned in [15], and suffers aliasing errors, a fact already identified in [16]. The most recent CNO has established a new state-of-the-art score in this field, which relies on U-Net as the core architecture and proceeds with several specified operations to circumvent aliasing errors. However, as a local operator, CNO falls short in capturing global information, which is of vital importance for functions. Another recently reported model, Mamba [17, 18], has also attracted a great deal of attention due to its capability in capturing global information with linear complexity. Unfortunately, the connection of state space model in Mamba to the integral kernel in NOs is far from intuitive. In addition, Mamba often produces artifacts [19] or pixel adhesion [20], raising the challenge of seeking continuous-discrete equivalence (CDE), as highlighted in [15]. Both these two have plagued the naive use of Mamba in the context of NO learning. In this work, we attempt to marry the current merits yet with the disadvantages suppressed. The practical contributions are threefold.

- We propose a novel integral form as the NO kernel, namely mamba integration, costing only $O(N)$ computational complexity and enabling the grasp of global function information. On that basis, with the local function feature further furnished by convolution, we present an avant-garde Mamba Neural Operator (MambaNO), behaving as a deep PDE sovler.
- Apart from proving that MambaNO is intrinsically a representation-equivalent neural operator in the sense of [21], we also provide a universality result to demonstrate that MambaNO can approximate continuous operators, fitting a large class of PDEs, to desired accuracy.
- We test MambaNO on a full set of benchmarks that span across massive PDEs ranging from linear elliptic and hyperbolic to nonlinear parabolic and hyperbolic equations, with potentially multiscale solutions. It is evidenced that MambaNO outperforms competing baselines on all benchmarks, both when testing in-distribution and in out-of-distribution cases, yet with reduced model parameters and computational costs. Codes are available [2].

Hence, we offer a new Mamba-based operator learning model incorporating more holistic features, with favorable properties in theory and excellent performance in practice.

## 2 Related Work

### 2.1 Learnable PDE Solvers

The development of deep learning has revitalized various fields [22]. In recent years, numerous investigations have been conducted on the application of learning networks to solve PDEs [23, 24]. A prevalent treatment involves initially encoding the data spatially, followed by the employment of diverse schemes for temporal evolution. For example, physics-informed approaches utilize PDE supervision to approximate the solution [25], commonly parameterized by neural networks, within a consolidated space-time domain. Yet, the accuracy is negatively correlated with solving efficiency. Moreover, retraining is required across different instances of a PDE.

To design more robust and efficient algorithms, Lu *et al.* [6] pioneeringly introduced the practical implementation of universal operator approximation theorem [5], which can further be integrated with prior knowledge of the system [26]. Independently, infinite-dimensional solution operators are approximated by iteratively learnable integral kernels [7]. Such kernel can be parameterized by message-passing [27], neural networks [28], attention mechanism [11, 29], convolution in Fourier domain [8] or in bandlimited function space [13]. Following this, we further present a novel form called as mamba integration, capturing global function information in $O(N)$ time complexity.

### 2.2 Alias-free Framework

Aligned with the discussion in [30], a neural operator or an operator learning architecture should hold the ability in managing functions as both inputs and outputs. Yet, in digital environments, engagement

---

and computation with functions are implemented via discrete representations, e.g., grid point values, cell averages, or generally coefficients of a specific basis, instead of the original continuous entities. Recently, the alias-free framework [15], in which the aliasing error is zero, was proposed. We provide a detailed definition in the Supplementary Material (SM) A.1. Within this framework, continuous functions should be consistently, or uniquely and stably, derived from the discrete counterparts, e.g., point evaluations, basis coefficients, etc., at any resolution, hammering at reconciling potential inconsistencies between operators and their discrete representations. Models that do not follow this framework, such as CNNs, may generate inconsistent outputs as the resolution varies, and thus are not Representation equivalent Neural Operators (ReNO), whose definition is given in SM A.2. Similarly, the property of continuous-discrete equivalence (CDE), proposed by [15] that measures the consistency between discrete and continuous representations, cannot be guaranteed. In addition, the high-frequency information introduced by point-wise activation functions also leads to similar inconsistencies [31, 20]. Therefore, FNO cannot be considered as a ReNO that meets the alias-free framework. More recently, CNO was elaborated as the first instance that conforms to the alias-free framework [13], paving the way for the implementation of our MambaNO.

## 3 Mamba Neural Operator

**Setting.** The core purpose here involves learning a mapping between two infinite-dimensional spaces via a finite array of observed input-output pairings from this correspondence. Generally, the problem concerned can be formally delineated as follows. Let $\mathcal{X} = H^r(D, \mathbb{R}^{d_{\mathcal{X}}})$ and $\mathcal{Y} = H^s(D, \mathbb{R}^{d_{\mathcal{Y}}})$ be Sobolev spaces of functions defined on 2-dimensional bounded domains $D = \mathbb{T}^2$ and $\mathcal{G}^\dagger : \mathcal{X} \to \mathcal{Y}$ be a non-linear map. By constructing a parametric map $\mathcal{G}_\theta$, the practice would be to build an approximation of $\mathcal{G}^\dagger$ from data pairs $\left(u_i, \mathcal{G}^\dagger(u_i)\right)_{i=1}^N \in \mathcal{X} \times \mathcal{Y}$, i.e.,

$$\mathcal{G}_\theta : \mathcal{X} \to \mathcal{Y}, \quad \theta \in \mathbb{R}^p, \tag{3.1}$$

with parameters from the finite-dimensional space $\mathbb{R}^p$ by seeking $\theta^\dagger \in \mathbb{R}^p$ so that $\mathcal{G}_{\theta\dagger} \approx \mathcal{G}^\dagger$. The practical learning of $\mathcal{G}_\theta$ can be naturally addressed through the empirical-risk minimization problem,

$$\min_{\theta \in \mathbb{R}^p} \mathbb{E}\|\mathcal{G}^\dagger(u) - \mathcal{G}_\theta(u)\|_{\mathcal{Y}}^2 \approx \min_{\theta \in \mathbb{R}^p} \frac{1}{N} \sum_{i=1}^N \|\mathcal{G}^\dagger(u_i) - \mathcal{G}_\theta(u_i)\|_{\mathcal{Y}}^2, \quad u \in \mathcal{X}. \tag{3.2}$$

Given that our methodology is conceived within the infinite-dimensional context, each finite-dimensional approximation enjoys a shared set of network parameters which are consistent in infinite environment devoid of approximations.

**Bandlimited Approximation.** Since the original Sobolev space is too large to allow for any form of continuous-discrete equivalence (A.1), we select a subspace $\mathcal{B}$ instead of the original $\mathcal{H}$. On that basis, the possibility of achieving equivalence between the underlying operator and its discrete representations would be basically guaranteed. In this respect, we elaborate the space of bandlimited functions defined by

$$\mathcal{B}_w(D) = \{f \in L^2(D) : \operatorname{supp}\widehat{f} \subseteq [-w, w]^2\}, \tag{3.3}$$

in which $w > 0$ denotes the frequency bound and $\widehat{f}$ represents the Fourier transform of $f$. Note that if a bandlimited function can approximate the original function with arbitrary precision (depending on $w$), then a bandlimited operator mapping between bandlimited functions can also approximate the original operator with arbitrary precision. In other words, for any $\varepsilon > 0$, there exist a $w$ and a continuous operator $\mathcal{G}_\theta : \mathcal{B}_w(D) \to \mathcal{B}_w(D)$ such that $\|\mathcal{G}^\dagger - \mathcal{G}_\theta\| < \varepsilon$, with $\|\cdot\|$ pertaining to the corresponding operator norm. In addition, let $P_w$ denote a certain Fourier projection, $P_w(g)$ is capable of discarding the high-frequency components higher than frequency $w$, where $g \in \mathcal{H}(D)$ is any function in that space.

**Definition of MambaNO.** As the underlying operator maps between the spaces of bandlimited functions, the operator approximation architecture shall be constructed in a structure-preserving fashion. That is to say, it is dedicated to form a corresponding mapping in-between bandlimited functions, thus respecting the CDE property. To that end, a Mamba Neural Operator (MambaNO) is engineered to approximate the operator $\mathcal{G}_\theta : \mathcal{B}_w(D) \to \mathcal{B}_w(D)$. Following the common paradigm of most NOs [12, 32, 33], our practical elaboration also lies in a compositional mapping,

$$\mathcal{G}_\theta := \mathcal{Q} \circ P_T(\eta_T(W_{T-1} + \mathcal{K}_{T-1} + b_{T-1})) \circ \cdots \circ P_1(\eta_1(W_0 + \mathcal{K}_0 + b_0)) \circ \mathcal{P}, \tag{3.4}$$

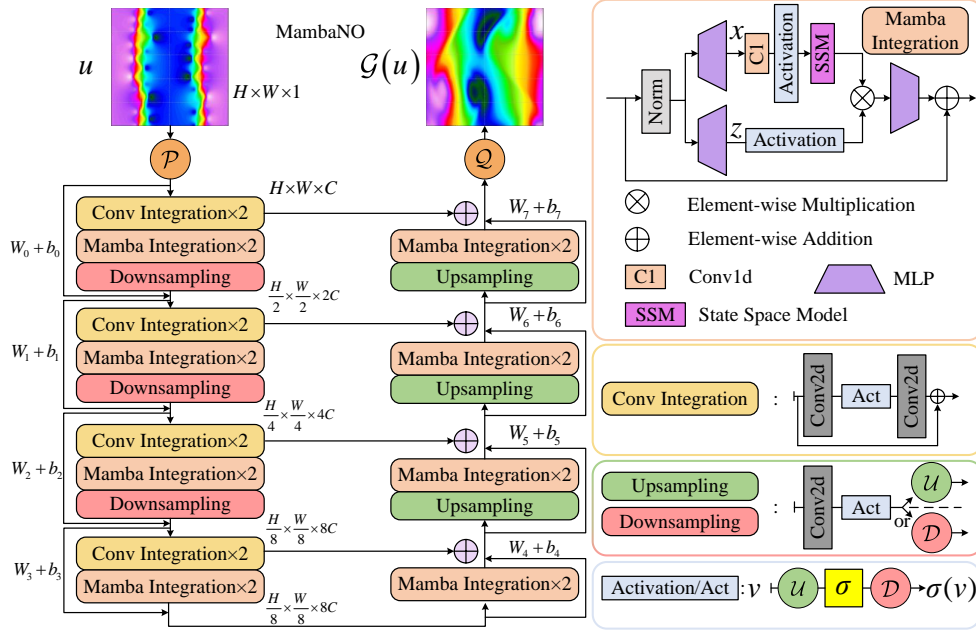

Figure 1: The overall architecture of MambaNO.

where

$$\mathcal{P} : \left\{ u \in \mathcal{B}_w(D, \mathbb{R}^{d_\mathcal{X}}) \right\} \rightarrow \left\{ v_0 \in \mathcal{B}_w(D, \mathbb{R}^{d_0}) \right\}, \tag{3.5a}$$

$$\mathcal{Q} : \left\{ v_T \in \mathcal{B}_w(D, \mathbb{R}^{d_T}) \right\} \rightarrow \left\{ \tilde{u} \in \mathcal{B}_w(D, \mathbb{R}^{d_\mathcal{Y}}) \right\}. \tag{3.5b}$$

are the lifting and projection mappings respectively, $W_t \in \mathbb{R}^{d_{t+1} \times d_t}$ are linear operators in a matrix form, $b_t : D \rightarrow \mathbb{R}^{d_{t+1}}$ are bias functions, $\eta_t$ are activation functions, and $P_t$ are either upsampling or downsampling operators in each layer. Note that all these operations are performed locally or pointwisely, following the principle of discretization invariance [34] to some extent. The general formulation of the kernel integral operator $\mathcal{K}_t : \{v_t : D \rightarrow \mathbb{R}^{d_t}\} \rightarrow \{v_{t+1} : D \rightarrow \mathbb{R}^{d_{t+1}}\}$ is parameterized by $\alpha$ such as:

$$(\mathcal{K}_t(v_t); \alpha)(x) = \int_D K_t(x, y, v_t(x), v_t(y)) v_t(y) \mathrm{d}y \quad \forall x, y \in D, \tag{3.6}$$

in which the parameter of kernel $K_t$ is learnt from given data. For instance, FNO [8, 35] employs convolution as the primary operation, while Transformer-based neural operators [11, 36] leverage attention mechanisms. In this work, a new integral form fitting Mamba architecture is crafted.

**Mamba Integration.** For simplicity of the exposition, we omit the subscript $t$ from Eq. (3.6) hereafter, which is originally used to denote the number of iterations during integration flow.

$$(\mathcal{K}_t(v_t); \alpha)(x) = \int_D K_t(x, y, v(x), v(y)) v(y) \mathrm{d}y. \tag{3.7}$$

Following the tradition of FNO [8, 37], we first assume that $K_t : \mathbb{R}^D \times \mathbb{R}^D \rightarrow \mathbb{R}^{d_{t+1} \times d_t}$ concerns little on the spatial variables $(v(x), v(y))$, but only on the input pair $(x, y)$. Then, we let

$$K_t(x, y) = Ce^{Ax} \cdot Be^{-Ay}, \tag{3.8}$$

where $A$, $B$ and $C$ are temporarily constant parameters, as for an easier deduction purpose. To further ensure a possible employment of the scanning pattern used in Mamba [17], we set the integration interval to $y \in (-\infty, x)$ instead of the entire definition domain $D$, hence Eq. (3.7) becomes

$$(\mathcal{K}_t(v_t); \alpha)(x) = \int_{-\infty}^{x} (Ce^{Ax} \cdot Be^{-Ay}) v(y) \mathrm{d}y. \tag{3.9}$$

Clearly, $Ce^{Ax}$ is independent of the integral variable $y$, from which Eq. (3.9) can be rewritten as:

$$(\mathcal{K}_t(v_t); \alpha)(x) = Ch(x), \text{with } h(x) = e^{Ax} \int_{-\infty}^{x} B(e^{-Ay})v(y)\mathrm{d}y. \qquad (3.10)$$

Furthermore, with simple operation of differential performed on $x$, we can get

$$h^{'}(x) = Ah(x) + Bv(x), \qquad (3.11)$$

which together with Eq. (3.10) leads to

$$
\begin{aligned}
h'(x) &= Ah(x) + Bv(x), \\
u(x) &= Ch(x),
\end{aligned}
\qquad (3.12)
$$

where $u(x) = (\mathcal{K}_t(v_t); \alpha)(x)$. More details of the deduction can be found in SM B.1. By now, we have seamlessly married the computation of kernel integral in Eq. (3.7) with a State Space Model (SSM) [38]. Drawing inspiration from the theory of continuous systems, the goal of Eq. (3.12) is to map a two-dimensional function, denoted as $v(x)$, to $u(x)$ through the hidden space $h(x)$. Within this context, $A$ serves as the evolution parameter, while $B$ and $C$ act as the projection parameters. To integrate Eq. (3.12) into deep learning paradigm, a discretization process is initially necessary. Note this transformation is crucial to align the model with the sampling rate of the underlying signal embodied in the input data, enabling computationally efficient operations. To address the drift and diffusion effects within most PDEs, unlike the Monte Carlo approximation used in conventional integrals, our approach employs the Scharfetter-Gummel method [39], which approximates the matrix exponential using Bernoulli polynomials, and can be formally defined as follows:

$$
\begin{aligned}
\bar{\mathbf{A}} &= \exp(\Delta A), \\
\bar{\mathbf{B}} &= (\Delta A)^{-1} \left( \exp(\Delta A) - \mathbf{I} \right) \cdot \Delta B,
\end{aligned}
\qquad (3.13)
$$

where $\Delta$ is a timescale parameter converting continuous parameters $A$ and $B$ into their discrete counterparts $\bar{\mathbf{A}}$ and $\bar{\mathbf{B}}$. The discrete representation of Eq. (3.12) can be formulated as follows:

$$
\begin{aligned}
h(x_k) &= \bar{\mathbf{A}}h(x_{k-1}) + \bar{\mathbf{B}}v(x_k), \\
u(x_k) &= Ch(x_k),
\end{aligned}
\qquad (3.14)
$$

More details can be found in SM A.3. Recall that our intention is to integrate the two-dimensional function in a scanning manner for integration, capturing global information with $O(N)$ complexity. In Eq. (3.14), $h(x)$, serving as the hidden space, encapsulates relevant information about the integrated points before $x$. Therefore, through Carleman bilinearization, we can construct a kernel to approximate the nonlinear state space evolution [40, 41], i.e., the output can be derived through global convolution:

$$
\begin{aligned}
\bar{\mathbf{K}} &= (C\bar{\mathbf{B}}, C\bar{\mathbf{A}}\bar{\mathbf{B}}, ..., C\bar{\mathbf{A}}^{(k-1)}\bar{\mathbf{B}}), \\
u(x_k) &= v(x_k) * \bar{\mathbf{K}},
\end{aligned}
\qquad (3.15)
$$

where $\bar{\mathbf{K}} \in \mathbb{R}^k$ denotes a structured convolution kernel and $k$ is the sampling points of input $v$.

**Convolution Integration.** Mamba integration enjoys a global receptive field, yet we believe that introducing local convolution integration would further bring benefits in capturing more holistic features. To commence with, the convolution integration for $\mathcal{K}_w : \mathcal{B}_w(D) \rightarrow \mathcal{B}_w(D)$ is defined as:

$$\mathcal{K}_w f(x) = \int_D \kappa_w(x - y)f(y)\mathrm{d}y = \sum_{i,j=1}^{k} \kappa_{ij} f(x - z_{ij}), \quad \forall x \in D, \qquad (3.16)$$

where $f \in \mathcal{B}_w$, $\kappa$ is a discrete kernel with size $k \in \mathbb{N}$, $z_{ij}$ is the resultant grid points. Thus, the convolution operator can be intuitively parameterized in physical space, deviating far from the treatments of Fourier transformation and then followed by matrix multiplication, as in FNO [8].

**Upsampling, Downsampling, and Activation Operators.** Ideal upsampling is recognized as not altering the continuous representation, simply transforming the original function to a larger bandwidth-limited space. In other words, just viewing the function from a band-limited space as belonging to a higher-bandwidth space does not actually change any values of the function. The upsampling operators for some $\overline{w} > w$ are defined as,

$$\mathcal{U}_{w,\overline{w}} : \mathcal{B}_w(D) \rightarrow \mathcal{B}_{\overline{w}}(D), \quad \mathcal{U}_{w,\overline{w}}f(x) = f(x), \quad \forall x \in D. \qquad (3.17)$$

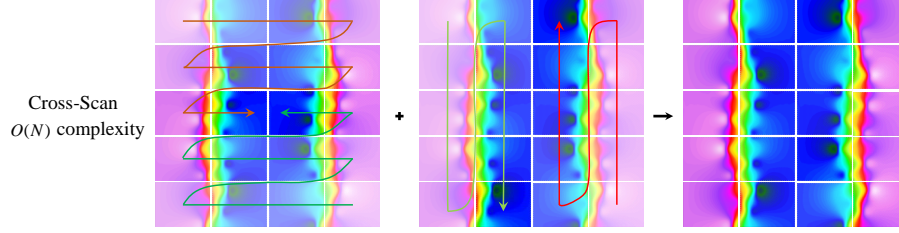

Cross-Scan
$O(N)$ complexity

**+**

**→**

Figure 2: The cross-scan operation integrates pixels from four directions with $O(N)$ complexity.

On the other side, for certain $\underline{w} < w$, the concerned function $f \in \mathcal{B}_w$ can be downsampled to lower band $\mathcal{B}_{\underline{w}}$ by setting $\mathcal{D}_{w,\underline{w}} : \mathcal{B}_w(D) \to \mathcal{B}_{\underline{w}}(D)$, defined by

$$\mathcal{D}_{w,\underline{w}}f(x) = \left(\frac{\underline{w}}{w}\right)^2 \int_D h_{\underline{w}}(x-y)f(y)\mathrm{d}y, \quad \forall x \in D, \tag{3.18}$$

where $h_{\underline{w}}$ is an interpolation sinc filter, playing the role of removing high-frequency components exceeding the lower band limit $\underline{w}$ and converting the function from a higher band-limited space $\mathcal{B}_w(D)$ to a lower band-limited space $\mathcal{B}_{\underline{w}}(D)$. As known, applying the activation function point-wise directly would break the band-limits of the underlying function space, potentially introducing arbitrarily higher-frequency information and causing aliasing errors [20]. We aim to apply the activation function within a sufficiently large band-limited $\overline{w}$ space to minimize the introduction of high-frequency information and thereby reduce aliasing errors. To this end, we define the activation operators in (3.4) as,

$$\eta_{w,\overline{w}} : \mathcal{B}_w(D) \to \mathcal{B}_w(D), \quad \eta_{w,\overline{w}}f(x) = \mathcal{D}_{\overline{w},w}(\sigma \circ \mathcal{U}_{w,\overline{w}}f)(x), \quad \forall x \in D. \tag{3.19}$$

Recall that the activation operator within a broader band-limited space is confined in a 'sandwich-like' structure, as shown in Eq. (3.4), with lifting $P$ and projection $Q$ lying at the edges and kernel integrations occupying the middle position.

---

**Algorithm 1**     SSM Block (Mamba Integration)

---

**Input:** $v(x)$, a continuous funtion with shape [sampling points $(N)$, dimension $(D)$]
**Parameters:** $A$, an evolution parameter; $\Delta$, a timescale parameter; $B$ and $C$, projection parameters
**Linear Projection Layer:** $\text{Linear}(\cdot)$
**Output:** $u(x)$, a function also with shape [sampling points, dimension]
1: $\Delta$, $B$, $C$ = Linear($v$), Linear($v$), Linear($v$)
2: $\bar{\mathbf{A}} = \exp(\Delta A)$
3: $\bar{\mathbf{B}} = (\Delta A)^{-1}(\exp(\Delta A) - \mathbf{I}) \cdot \Delta B$

---

4: $u(x) = SSM(\bar{\mathbf{A}}, \bar{\mathbf{B}}, C)(v(X))$, $X$ is a discrete sequence that contains $x_1, x_2, \cdots, x_N$.
     4.1: $h(x_k) = \bar{\mathbf{A}}h(x_{k-1}) + \bar{\mathbf{B}}v(x_k)$
     4.2: $u(x_k) = Ch(x_k)$
     4.3: $u(x) = [u(x_1), u(x_2), \cdots, u(x_N)]$

---

5: **return** $u(x)$

---

**MambaNO Architecture.** Given bandlimited functions as inputs and outputs, all the concerned ingredients can be assembled in a U-shaped operator architecture, which is graphically given in Fig. 1. As seen, the input function, say $u \in \mathcal{B}_w(D, \mathbb{R}^{d_x})$ is first lifted and then processed through a series of layers. Five main layers are used, i.e., convolution integration (3.16), Mamba integration (3.15), activation layer (3.19), upsampling layers (3.17), and downsampling layers (3.18). Each layer is fed with a band-limited function, and another band-limited function holding the same band returns as the output. As the entire operation flow runs solely in the channel width, the underlying bandlimits are confirmed since the spatial resolution is left unchanged. Thus, MambaNO assumes a function input and throws it into a set of encoders, where the input is space-wise downsampled but channel-wise widened. Then, the function is passed through a set of decoders, where the channel width is shrunk, yet the space resolution is enlarged. In the meantime, the encoder and decoder layers sharing the same spatial resolution and bandlimits are coupled with a resnet architecture. Thus, as we go deeper into the encoder flow, transferring high-frequency information via skip connections is allowed, avoiding

which to be totally filtered out with the sinc operation. In other words, the high-frequency information is not just produced by the activation function but also altered through the intermediate modules.

Practically, the constant parameters $A$, $B$, and $C$ are better learned from the data, allowing attractive model adaptability. This together with the shifted integral interval, as shown in Eq. (3.9), enables cross-scan operation within the state space. As illustrated in Fig. 2, we choose to unfold sampling points into sequences along with rows and columns and then proceed with scanning along four different directions, i.e., top-left to bottom-right, bottom-right to top-left, top-right to bottom-left, and bottom-left to top-right. These sequences are further processed by the SSM block for kernel integration, ensuring that information from various directions is thoroughly scanned, thus capturing diverse function features. Afterwards, the sequences from the four directions are merged, restoring the output function. The pseudo-code for the SSM block is presented in Algorithm 1.

**Continuous-discrete Equivalence for MambaNO.** We have defined MambaNO (3.4) as an mapping operator in-between bandlimited functions and that runs in a scanning pattern within a state space. In practice, like any other computational methods, MambaNO shall be performed in a discrete manner, with discretized variants of individual layers specified in SM A.3. Given the practical implementations of each elementary block, i.e., convolution, up- and down-sampling, activation, etc., we then prove the following proposition, whose details are given in SM B.2:

**Proposition 3.1** *Mamba Neural Operator $\mathcal{G} : \mathcal{B}_w(D, \mathbb{R}^d \mathcal{X}) \to \mathcal{B}_w(D, \mathbb{R}^d \mathcal{Y})$ is a Representation equivalent Neural Operator (ReNO). That is, MambaNO enjoys the CDE property.*

More details on the notion of ReNOs can be found in SM A.2. As a ReNO, MambaNO is representation-equivalent, allowing its migration between grids of different scales, yet with little aliasing errors. This property, which can also be called resolution-discrete invariance, as a significant characteristic of the neural operator, is highlighted in Ref. [8].

**Complexity Analysis of MambaNO.** Given a discrete counterpart $u \in R^{H \times W \times D}$ of a two-dimensional continuous function, one can reshape it to get $u \in R^{N \times D}$ with $N = H \times W$. For simplicity, we assume the sequence dimensions of $\Delta, A, B, C$ are all $D$. As shown in step 1 of Algorithm 1, the complexity of generating three learnable projections is $O(3ND^2)$. The dicretization steps 2 and 3 involve four matrix multiplications, which cost $O(4ND^2)$. Step 4 is the state space model with $O(N)$ complexity [42]. In summary, the computations of Algorithm 1 are all linear with the sequence length, i.e., $O(N)$. Please refer to SM C for the time complexity of other models.

# 4 Experiments and Analysis

## 4.1 Experimental Settings

**Training Details and Baselines.** For fairness and reliability, all experiments are consistently conducted on standardized platform with an NVIDIA RTX 3090 GPU and 2.40GHz Intel(R) Xeon(R) Silver 4210R CPU. Several well-known PDE solvers are used as the competing baselines, such as CNO [13], FNO [8], DeepONet (DON) [6], Galerkin Transformer (GT) [14], as well as the very typical ResNet[43] and U-Net[44] architectures.

**Representative PDE Benchmarks (RPB).** As a standard set of benchmarks for machine learning of PDEs, RBP focuses solely on two-dimensional PDEs since conventional numerical methods have already yielded quite pleasing outcomes on one-dimensional functions; however, procuring training data for those in three dimensions or higher is immensely cost-prohibitive. With these considerations in mind, RBP covers Poisson Equation, Wave Equation, Transport Equation, Allen-Cahn Equation, Navier-Stokes Eqns, Darcy flow, and Flow past airfoils, which are defined on Cartesian domains. We have roughly listed the related information of the equations in SM D.

## 4.2 In and Out-of-distribution Results

The test results for both the in- and out-of-distribution evaluations from all competing models are shown in Table 1. Specific for in-distribution experiments, it can be easily observed that, among all competitors, except for the Allen-Cahn equation, CNO enjoys an evident superiority compared to others. Moreover, our MambaNO, benefiting from the introduction of both global and local integrations, outperforms even further. Taking the Poisson equation as an instance, CNO performs twenty times better than FNO, while MambaNO further reduces the error by one-third. On the other

Table 1: Relative median $L^1$ test errors for various benchmarks and models.

|  | In/Out | GT | Unet | ResNet | DON | FNO | CNO | MambaNO |
|---|---|---|---|---|---|---|---|---|
| **Poisson Equation** | In | 4.09% | 1.05% | 0.63% | 19.07% | 7.35% | 0.31% | 0.17% |
|  | Out | 3.47% | 1.55% | 1.34% | 11.18% | 8.62% | 0.33% | 0.21% |
| **Wave Equation** | In | 0.91% | 0.96% | 0.70% | 1.43% | 0.65% | 0.40% | 0.38% |
|  | Out | 1.97% | 2.24% | 2.50% | 3.12% | 1.95% | 1.29% | 1.22% |
| **Smooth Transport** | In | 1.18% | 0.59% | 0.47% | 1.38% | 0.34% | 0.29% | 0.26% |
|  | Out | 666.07% | 2.97% | 2.73% | 119.61% | 1.97% | 0.35% | 0.34% |
| **Discontinuous Transport** | In | 1.70% | 1.44% | 1.41% | 6.35% | 1.26% | 1.11% | 1.08% |
|  | Out | 27270.96% | 1.62% | 1.54% | 140.73% | 3.47% | 1.31% | 1.21% |
| **Allen-Cahn Equation** | In | 1.30% | 1.38% | 2.36% | 22.97% | 0.87% | 0.91% | 0.72% |
|  | Out | 3.03% | 3.28% | 3.91% | 20.75% | 2.18% | 2.33% | 2.11% |
| **Navier-Stokes Equation** | In | 4.61% | 4.94% | 4.10% | 12.95% | 3.97% | 3.07% | 2.74% |
|  | Out | 17.23% | 16.98% | 15.04% | 23.39% | 14.89% | 10.94% | 5.95% |
| **Darcy Flow** | In | 0.86% | 0.54% | 0.42% | 1.13% | 0.80% | 0.38% | 0.33% |
|  | Out | 1.17% | 0.64% | 0.60% | 1.61% | 1.11% | 0.50% | 0.44% |
| **Compressible Euler** | In | 2.33% | 0.72% | 1.89% | 2.15% | 0.49% | 0.39% | 0.34% |
|  | Out | 3.14% | 0.91% | 2.20% | 3.08% | 0.74% | 0.63% | 0.61% |

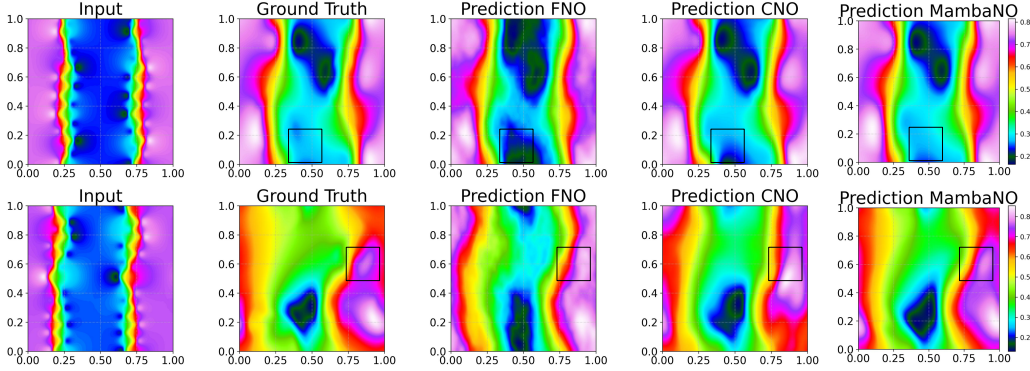

Figure 3: Visual predictions on representative in- (top row) and out-of-distribution (bottom row).

hand, we can see that UNet and ResNet also behave well in some cases. This not only verifies the feasibility of convolution as kernel integration, but also shows that deep learning can directly ascertain the trajectory of an equation family from the data. However, the limitations are also evident, lying specifically in the fact that little consideration is given on the alias-free framework. Therefore, when migrating to out-distribution experiments, the experimental results concerned show a significant decline. We can see that in the out-distribution experiments of the Navier-Stokes equation, their performance drops more than that of CNO, which is another instance that adheres to the alias-free modeling. MambaNO again sets a new state-of-the-art score in this case, which not only follows this framework, but also benefits from the global integration. More intuitively, Fig. 3 plots the in-distribution and out-of-distribution visualization of the Navier-Stokes equation, taking the three best-performing models, e.g., FNO, CNO, and MambaNO, as examples. Our first observation is that, whether in the in-distribution or out-distribution environments, the predictions of MambaNO are the closest to the ground truth. The regions within the black box provide the most clear comparisons. Note that FNO is limited by the pointwise activation function that introduces aliasing errors, and CNO is limited by the convolution integral paying little attention to global information, while MambaNO has made improvements on both these issues.

### 4.3 Resolution Invariance

In the left and central segments of Fig. 4, we compare UNet, FNO, CNO, and MambaNO vis-a-vis the metric of varying errors across resolutions, which is an important property for robust operator learning. The selected equation is the Navier-Stokes benchmark. In this figure, we can see that both UNet and FNO are practically not discrete invariant. With resolution changing, a sharp decline

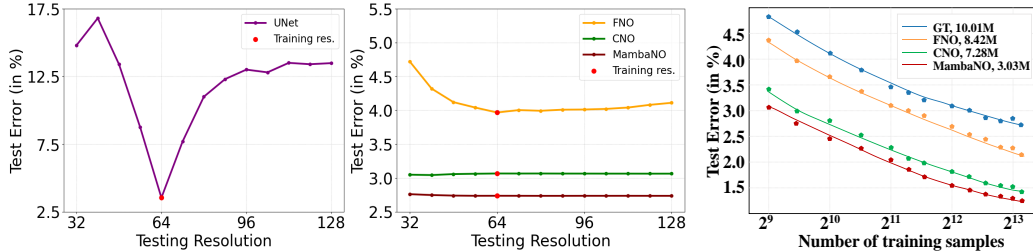

Figure 4: Left & Center: Test errors vs. Resolutions. Right: Errors vs. Training samples.

happens in the performance of UNet. For FNO, when the resolution is less than 64, there is a 25% decrease in performance. As resolution increases, a modest fluctuation in errors also occurs. Thanks to the alias-free framework, CNO and MambaNO enjoy much flatter curves along with different resolutions, validating the advantage of respecting CDE property. Between these two, MambaNO further enjoys much lower errors, which is again attributed to the combination of convolution and mamba integration, introducing the more holistic function features.

Variations in model performance against different data scales are also evaluated on Navier-Stokes. The right segment of Fig. 4 provides the test errors in the log domain. As shown, MambaNO consistently achieves optimal performance compared to GT, FNO, and CNO, regardless of the number of samples used. More encouragingly, the performance lead is yet achieved with the least model parameters.

## 5 Ablation Study

To affirm the benefit from combining both global and local information, Table 2 discloses results pertaining to the ablation of Mamba and Convolution integration. Based on our final configuration, the mamba integration is replaced with convolution integral, thus obtaining baseline 1 with pure local terms. Similarly, we can also obtain baseline 3, which holds pure mamba integration. To ensure a fair comparison, the parameters of all three competitors are configured at the same level. As expected, due to the lack of partial information, the performance of pure convolution integration and pure Mamba integration drops by 14% and 32%, respectively. In terms of efficiency, while pure convolution treatment enjoys less inference time and FLOPs in this case, it cannot be ensured in practice since more network layers are needed to enlarge the receptive field. Due to space limitations, we have placed more ablation experiments in SM E.

Table 2: The ablation results by using different components. * indicates our default choice.

| Configuration | Test errors ↓ | Time ↓ | Params ↓ | FLOPs ↓ |
|---|---|---|---|---|
| 1. Pure Convolution integration | $1.14 \times$ | $0.83 \times$ | $0.98 \times$ | $0.79 \times$ |
| 2. Mamba+Convolution integration* | $1.00 \times$ | $1.00 \times$ | $1.00 \times$ | $1.00 \times$ |
| 3. Pure Mamba integration | $1.32 \times$ | $1.14 \times$ | $1.02 \times$ | $1.19 \times$ |

## 6 Conlusion

We propose MambaNO, a novel neural operator for solving PDEs, which incorporates Mamba integration and convolution integration to capture global and local function information holistically, yet in linear complexity. The basic design principle is to marry the integral kernel with the state space model. On that basis, with respect to spaces of bandlimited functions, CDE property is naturally satisfied to authentically assimilate the innate operators, as opposed to the discrete representation surrogation. A suite of experiments conducted on representative PDE benchmarks demonstrate that MambaNO outperforms the recent baselines, such as GT, FNO, and CNO, on many practical metrics namely test errors, running efficiency, resolution invariance, out-of-distribution generalization, and data scaling. **Limitation:** We have presented MambaNO for operators in a fundamental two-dimensional Cartesian domain. The expansion to three-space dimensions is theoretically intuitive, but computationally laborious. This intensifies the computational load for global integration, which inherently escalates the model parameters.

## Acknowledgements

This work was supported in part by the Pioneer and Leading Goose R&D Program of Zhejiang under Grant 2023C01241, the National Natural Science Foundation of China under Grant 62276232, the Key Program of Natural Science Foundation of Zhejiang Province under Grant LZ24F030012 and Zhejiang Students' Technology and Innovation Program under Grant 2024R403B071.

## References

[1] Kamyar Azizzadenesheli, Nikola Kovachki, Zongyi Li, Miguel Liu-Schiaffini, Jean Kossaifi, and Anima Anandkumar. Neural operators for accelerating scientific simulations and design. *Nature Reviews Physics*, pages 1–9, 2024.

[2] Alexander R Cobb, René Dommain, Kimberly Yeap, Cao Hannan, Nathan C Dadap, Bodo Bookhagen, Paul H Glaser, and Charles F Harvey. A unified explanation for the morphology of raised peatlands. *Nature*, 625(7993):79–84, 2024.

[3] Kevin Course and Prasanth B Nair. State estimation of a physical system with unknown governing equations. *Nature*, 622(7982):261–267, 2023.

[4] Ralf Konietzka, Alyssa A Goodman, Catherine Zucker, Andreas Burkert, João Alves, Michael Foley, Cameren Swiggum, Maria Koller, and Núria Miret-Roig. The radcliffe wave is oscillating. *Nature*, pages 1–2, 2024.

[5] Tianping Chen and Hong Chen. Universal approximation to nonlinear operators by neural networks with arbitrary activation functions and its application to dynamical systems. *IEEE Transactions on Neural Networks*, 6(4):911–917, 1995.

[6] Lu Lu, Pengzhan Jin, Guofei Pang, Zhongqiang Zhang, and George Em Karniadakis. Learning nonlinear operators via deeponet based on the universal approximation theorem of operators. *Nature Machine Intelligence*, 3(3):218–229, 2021.

[7] Anima Anandkumar, Kamyar Azizzadenesheli, Kaushik Bhattacharya, Nikola Kovachki, Zongyi Li, Burigede Liu, and Andrew Stuart. Neural operator: Graph kernel network for partial differential equations. In *International Conference on Learning Representations*, 2020.

[8] Zongyi Li, Nikola Borislavov Kovachki, Kamyar Azizzadenesheli, Kaushik Bhattacharya, Andrew Stuart, Anima Anandkumar, et al. Fourier neural operator for parametric partial differential equations. In *International Conference on Learning Representations*, 2020.

[9] Boris Bonev, Thorsten Kurth, Christian Hundt, Jaideep Pathak, Maximilian Baust, Karthik Kashinath, and Anima Anandkumar. Spherical fourier neural operators: Learning stable dynamics on the sphere. In *International Conference on Machine Learning*, pages 2806–2823. PMLR, 2023.

[10] Gege Wen, Zongyi Li, Kamyar Azizzadenesheli, Anima Anandkumar, and Sally M Benson. U-fno—an enhanced fourier neural operator-based deep-learning model for multiphase flow. *Advances in Water Resources*, 163:104180, 2022.

[11] Zijie Li, Kazem Meidani, and Amir Barati Farimani. Transformer for partial differential equations' operator learning. *Transactions on Machine Learning Research*, 2022.

[12] Zhongkai Hao, Zhengyi Wang, Hang Su, Chengyang Ying, Yinpeng Dong, Songming Liu, Ze Cheng, Jian Song, and Jun Zhu. Gnot: A general neural operator transformer for operator learning. In *International Conference on Machine Learning*, pages 12556–12569. PMLR, 2023.

[13] Bogdan Raonic, Roberto Molinaro, Tim De Ryck, Tobias Rohner, Francesca Bartolucci, Rima Alaifari, Siddhartha Mishra, and Emmanuel de Bézenac. Convolutional neural operators for robust and accurate learning of pdes. *Advances in Neural Information Processing Systems*, 36, 2024.

[14] Shuhao Cao. Choose a transformer: Fourier or galerkin. *Advances in Neural Information Processing Systems*, 34:24924–24940, 2021.

[15] Francesca Bartolucci, Emmanuel de Bezenac, Bogdan Raonic, Roberto Molinaro, Siddhartha Mishra, and Rima Alaifari. Representation equivalent neural operators: a framework for alias-free operator learning. *Advances in Neural Information Processing Systems*, 36, 2024.

[16] VS Fanaskov and Ivan V Oseledets. Spectral neural operators. In *Doklady Mathematics*, volume 108, pages S226–S232. Springer, 2023.

[17] Lianghui Zhu, Bencheng Liao, Qian Zhang, Xinlong Wang, Wenyu Liu, and Xinggang Wang. Vision mamba: Efficient visual representation learning with bidirectional state space model. *arXiv preprint arXiv:2401.09417*, 2024.

[18] Rui Xu, Shu Yang, Yihui Wang, Bo Du, and Hao Chen. A survey on vision mamba: Models, applications and challenges. *arXiv preprint arXiv:2404.18861*, 2024.

[19] Zehao Yu, Anpei Chen, Binbin Huang, Torsten Sattler, and Andreas Geiger. Mip-splatting: Alias-free 3d gaussian splatting. *arXiv preprint arXiv:2311.16493*, 2023.

[20] Tero Karras, Miika Aittala, Samuli Laine, Erik Härkönen, Janne Hellsten, Jaakko Lehtinen, and Timo Aila. Alias-free generative adversarial networks. *Advances in Neural Information Processing Systems*, 34:852–863, 2021.

[21] Francesca Bartolucci, Emmanuel de Bézenac, Bogdan Raonić, Roberto Molinaro, Siddhartha Mishra, and Rima Alaifari. Are neural operators really neural operators? frame theory meets operator learning. *SAM Research Report*, 2023, 2023.

[22] Yueqian Quan, Honghui Xu, Renfang Wang, Qiu Guan, and Jianwei Zheng. Orsi salient object detection via progressive semantic flow and uncertainty-aware refinement. *IEEE Transactions on Geoscience and Remote Sensing*, 62:5608013 –5608025, 2024.

[23] Lu Lu, Xuhui Meng, Zhiping Mao, and George Em Karniadakis. Deepxde: A deep learning library for solving differential equations. *SIAM review*, 63(1):208–228, 2021.

[24] George Em Karniadakis, Ioannis G Kevrekidis, Lu Lu, Paris Perdikaris, Sifan Wang, and Liu Yang. Physics-informed machine learning. *Nature Reviews Physics*, 3(6):422–440, 2021.

[25] Zongyi Li, Hongkai Zheng, Nikola Kovachki, David Jin, Haoxuan Chen, Burigede Liu, Kamyar Azizzadenesheli, and Anima Anandkumar. Physics-informed neural operator for learning partial differential equations. *ACM/JMS Journal of Data Science*, 2021.

[26] Sifan Wang, Hanwen Wang, and Paris Perdikaris. Learning the solution operator of parametric partial differential equations with physics-informed deeponets. *Science Advances*, 7(40): eabi8605, 2021.

[27] Justin Gilmer, Samuel S Schoenholz, Patrick F Riley, Oriol Vinyals, and George E Dahl. Neural message passing for quantum chemistry. In *International Conference on Machine Learning*, pages 1263–1272. PMLR, 2017.

[28] Zongyi Li, Nikola Kovachki, Kamyar Azizzadenesheli, Burigede Liu, Andrew Stuart, Kaushik Bhattacharya, and Anima Anandkumar. Multipole graph neural operator for parametric partial differential equations. *Advances in Neural Information Processing Systems*, 33:6755–6766, 2020.

[29] Wenhui Peng, Zelong Yuan, Zhijie Li, and Jianchun Wang. Linear attention coupled fourier neural operator for simulation of three-dimensional turbulence. *Physics of Fluids*, 35(1), 2023.

[30] Nikola Kovachki, Zongyi Li, Burigede Liu, Kamyar Azizzadenesheli, Kaushik Bhattacharya, Andrew Stuart, and Anima Anandkumar. Neural operator: Learning maps between function spaces with applications to pdes. *Journal of Machine Learning Research*, 24(89):1–97, 2023.

[31] Hagay Michaeli, Tomer Michaeli, and Daniel Soudry. Alias-free convnets: fractional shift invariance via polynomial activations. In *Proceedings of the IEEE/CVF Conference on Computer Vision and Pattern Recognition*, pages 16333–16342, 2023.

[32] Xiaoyu Zhao, Xiaoqian Chen, Zhiqiang Gong, Weien Zhou, Wen Yao, and Yunyang Zhang. Recfno: a resolution-invariant flow and heat field reconstruction method from sparse observations via fourier neural operator. *International Journal of Thermal Sciences*, 195:108619, 2024.

[33] Makoto Takamoto, Timothy Praditia, Raphael Leiteritz, Daniel MacKinlay, Francesco Alesiani, Dirk Pflüger, and Mathias Niepert. Pdebench: An extensive benchmark for scientific machine learning. *Advances in Neural Information Processing Systems*, 35:1596–1611, 2022.

[34] Samira Kabri, Tim Roith, Daniel Tenbrinck, and Martin Burger. Resolution-invariant image classification based on fourier neural operators. In *International Conference on Scale Space and Variational Methods in Computer Vision*, pages 236–249. Springer, 2023.

[35] Ning Liu, Siavash Jafarzadeh, and Yue Yu. Domain agnostic fourier neural operators. *Advances in Neural Information Processing Systems*, 36, 2024.

[36] Min Wei and Xuesong Zhang. Super-resolution neural operator. In *Proceedings of the IEEE/CVF Conference on Computer Vision and Pattern Recognition*, pages 18247–18256, 2023.

[37] Zongyi Li, Daniel Zhengyu Huang, Burigede Liu, and Anima Anandkumar. Fourier neural operator with learned deformations for pdes on general geometries. *Journal of Machine Learning Research*, 24(388):1–26, 2023.

[38] Albert Gu, Karan Goel, and Christopher Re. Efficiently modeling long sequences with structured state spaces. In *International Conference on Learning Representations*, 2021.

[39] AA Kulikovsky. A more accurate scharfetter-gummel algorithm of electron transport for semiconductor and gas discharge simulation. *Journal of computational physics*, 119(1):149–155, 1995.

[40] Yousef Saad. Analysis of some krylov subspace approximations to the matrix exponential operator. *SIAM Journal on Numerical Analysis*, 29(1):209–228, 1992.

[41] Yoshinobu Kawahara. Dynamic mode decomposition with reproducing kernels for koopman spectral analysis. *Advances in neural information processing systems*, 29, 2016.

[42] Albert Gu and Tri Dao. Mamba: Linear-time sequence modeling with selective state spaces. *arXiv preprint arXiv:2312.00752*, 2023.

[43] Kaiming He, Xiangyu Zhang, Shaoqing Ren, and Jian Sun. Deep residual learning for image recognition. In *Proceedings of the IEEE conference on Computer Vision and Pattern Recognition*, pages 770–778, 2016.

[44] Olaf Ronneberger, Philipp Fischer, and Thomas Brox. U-net: Convolutional networks for biomedical image segmentation. In *Medical image computing and computer-assisted intervention–MICCAI 2015: 18th international conference, Munich, Germany, October 5-9, 2015, proceedings, part III 18*, pages 234–241. Springer, 2015.

[45] Zongyi Li, Nikola Kovachki, Chris Choy, Boyi Li, Jean Kossaifi, Shourya Otta, Mohammad Amin Nabian, Maximilian Stadler, Christian Hundt, Kamyar Azizzadenesheli, et al. Geometry-informed neural operator for large-scale 3d pdes. *Advances in Neural Information Processing Systems*, 36, 2024.

# Supplementary Material
## Alias-Free Mamba Neural Operator

## A  Definition

### A.1  Operator Aliasing Error

**Setting.** Let $U : \text{Dom } U \subseteq \mathcal{H} \to \mathcal{K}$ be an operator between two dissociable Hilbert spaces, and let $\Psi = \{\psi_i\}_{i \in I}$ and $\Phi = \{\phi_k\}_{k \in K}$ be frame sequences for $\mathcal{H}$ and $\mathcal{K}$, respectively, with synthesis terms $T_\psi$ and $T_\phi$. Their closed linear spans are represented by $\mathcal{M}_\Psi := \overline{\text{span}}\{\psi_i : i \in I\}$ and $\mathcal{M}_\Phi := \overline{\text{span}}\{\phi_k : k \in K\}$. The pseudo-inverses $T_\Psi^\dagger$ and $T_\Phi^\dagger$, initially given on $\mathcal{M}_\Psi$ and $\mathcal{M}_\Phi$, respectively, can in practice be extended to the entire Hilbert spaces, i.e. $T_\Psi^\dagger : \mathcal{H} \to \ell^2(I)$ and $T_\Phi^\dagger : \mathcal{K} \to \ell^2(K)$. "Operator aliasing" describes the relationship established by identifying the continuous operator $U$ and its discrete counterpart $u : \ell^2(I) \to \ell^2(K)$, which is determined by the input and output frame sequences $(\Psi, \Phi)$. Then, we can build the operator $T_\Phi \circ u \circ T_\Psi^\dagger : \mathcal{H} \to \mathcal{K}$, whose definition clearly depends on the choices of the frame sequences that we make on the continuous level. Any mapping $u$ can be interpreted as a discrete representation of an underlying continuous operator $U$. We now give the definition of the Operator Aliasing Error:

**Definition: Operator Aliasing Error.** *The aliasing error operator $\varepsilon(U, u, \Psi, \Phi) : \text{Dom } U \subseteq \mathcal{H} \to \mathcal{K}$ is given by*

$$\varepsilon(U, u, \Psi, \Phi) = U - T_\Phi \circ u \circ T_\Psi^\dagger, \tag{A.1}$$

*and the corresponding scalar error is $\|\varepsilon(U, u, \Psi, \Phi)\|$, with $\|\cdot\|$ denoting the operator norm.*

An aliasing error of zero implies that the operator $U$ can be perfectly represented by first discretizing the function with $T_\Psi^\dagger$, applying $u$, and then reconstructing with $T_\Phi$. If the aliasing error is zero, we say that $(U, u, \Psi, \Phi)$ satisfies a **continuous-discrete equivalence (CDE)**, implying that accessing the discrete representation $u$ is exactly the same as accessing the underlying continuous operator $U$.

**How does such error occur? And how should we avoid it?** Consider the operator $U(f) = |f|^2$ as an operator from $\mathcal{B}_\Omega$ to $\mathcal{B}_{2\Omega}$. The choice to discretize inputs and outputs on the same grid $\left\{\frac{n}{2\Omega}\right\}_{n \in \mathbb{Z}}$ corresponds to choosing $\Psi = \Phi = \{\text{sinc}(2\Omega x - n)\}_{n \in \mathbb{Z}}$, and to defining the discrete mapping $u \colon \ell^2(\mathbb{Z}) \to \ell^2(\mathbb{Z})$ by $u(v) = T_\Psi^\dagger \circ U \circ T_\Psi(v) = v \odot \overline{v}$, where $\odot$ denotes the entrywise product.

Then, for every $f \in \mathcal{B}_\Omega$ such that $U(f) \in \mathcal{B}_{2\Omega} \setminus \mathcal{B}_\Omega$, we have

$$\varepsilon(U, u, \Psi, \Phi)(f) = U(f) - T_\Phi \circ T_\Phi^\dagger(U(f)) = U(f) - \mathcal{P}_{\mathcal{B}_\Omega}(U(f)) \neq 0. \tag{A.2}$$

Aliasing error occurs in this process, as $U$ introduces new frequencies that exceed the bandwidth $\omega$, and these high frequencies cannot be represented by the sequence framework $\Psi = \Phi = \{\text{sinc}(2\Omega x - n)\}_{n \in \mathbb{Z}}$. We can rectify this by sampling the output functions on a grid with twice the resolution of the input grid. This corresponds to choosing $\Phi = \{\text{sinc}(4\Omega x - n)\}_{n \in \mathbb{Z}}$ and to defining $u = T_\Phi^\dagger \circ U \circ T_\Psi$, which simply maps samples from grid points $\left\{\frac{n}{2\Omega}\right\}_{n \in \mathbb{Z}}$ into squared samples from the double resolution grid $\left\{\frac{n}{4\Omega}\right\}_{n \in \mathbb{Z}}$. This effectively removes aliasing since the equality $U = T_\Phi \circ u \circ T_\Psi^\dagger$ is satisfied. Furthermore, sampling the input and output functions with arbitrarily higher sampling rate, i.e., representing the functions with respect to the system $\{\text{sinc}(2\overline{\Omega} x - n)\}_{n \in \mathbb{Z}}$ with $\overline{\Omega} > 2\Omega$, yields no aliasing error since $\{\text{sinc}(2\overline{\Omega} x - n)\}_{n \in \mathbb{Z}}$ constitutes a frame for $\mathcal{B}_{2\Omega} \supseteq \mathcal{B}_\Omega$. The activation function that introduces high-frequency information can serve as a prime example of our above content.

We can determine different discretized versions $u$ and $u'$ of the continuous operator $U$ by choosing different sequence frameworks $\Psi, \Phi$ and $\Psi', \Phi'$, respectively. Therefore, the consistency between operations $u$ and $u'$ can be evaluated using the following error function.

**Definition: Representation equivalence error.** *The representation equivalence error function $\tau(u, u') : \ell^2(I) \to \ell^2(K)$, defined as:*

$$\tau(u, u') = u - T_\Phi^\dagger \circ T_{\Phi'} \circ u' \circ T_{\Psi'}^\dagger \circ T_\Psi, \tag{A.3}$$

*and the corresponding scalar error is $\|\tau(u, u')\|$, with $\|\cdot\|$ denoting the operator norm.*

## A.2 Representation equivalent Neural Operators

**Representation equivalent Neural Operators.** $(U, u)$ *is a Representation equivalent Neural Operator if for every frame-sequence pair* $(\Psi, \Phi)$ *that satisfies* $Dom\ U \subseteq \mathcal{M}_\Psi$ *and* $Ran\ U \subseteq \mathcal{M}_\Phi$, *no aliasing errors occur, i.e., the aliasing error operator is identical to zero:*

$$\varepsilon(U, u, \Psi, \Phi) = 0 \quad or \quad \varepsilon(U, u) = 0. \tag{A.4}$$

In this case, all discrete representations $u(\Psi, \Phi)$ are equivalent, demonstrating that they uniquely determine the same underlying operator $U$, whenever a continuous-discrete equivalence property holds at the level of the function spaces. This also implies that we can unambiguously determine the discrete $u(\Psi, \Phi) = T_\Phi^\dagger \circ U \circ T_\Psi$ consistent with the continuous counterpart $U$ by fixing the representations $\Psi, \Phi$ concerned with the input and output functions. In practice, we shall have access to different representation grids of the inputs and outputs, and their associated representation error is zero, e.g., $\tau(u, u') = 0$ (please refer to SM. A.3), where $u'$ denotes $u(\Psi', \Phi')$. Now, we establish a connection between aliasing and representation equivalence.

In addition, assume that $u$ and $u'$ are separate discrete forms of $U$, concerning to frame sequences $(\Psi, \Phi)$ and $(\Psi', \Phi')$, respectively. Then, the transformation between these discrete forms can be implemented through the formula:

$$u(\Psi', \Phi') = T_{\Phi'}^\dagger \circ T_\Phi \circ u(\Psi, \Phi) \circ T_\Psi^\dagger \circ T_{\Psi'}. \tag{A.5}$$

## A.3 Discrete layers for MambaNO

**Setting.** Assume that the discrete multichannel signal obtained after uniformly sampling a continuous signal $a \in \mathcal{B}_w(D)$ is: $a_s \in \mathbb{R}^{s \times s \times d}$, then $a_s[i, j, c]$ refers to the $(i, j)$-th coordinate of the $c$-th channel of the signal, where $i, j = 1...s$ and $c = 1...d$.

**Convolution Integral.** First, we consider the convolution integral of a single channel, that is $c = 1$. And, the discrete version of the convolution kernel corresponding to Eq. (3.16) is $K_w \in \mathbb{R}^{k \times k}$. In order to ensure that the input and output signals have the same spatial dimension $s \times s$, we choose to perform zero-padding of length $\tilde{k}$ on the original discrete signal, where $\tilde{k} = (k - 1)/2$. Then the discrete form of the convolution integral for a single channel is:

$$\mathcal{K}(a) = (a_s \star K_w)[i, j] = \sum_{m,n=-\tilde{k}}^{\tilde{k}} K_w[m, n] \cdot \tilde{a}_s[i - m, j - n], \quad i, j = 1 \ldots s, \tag{A.6}$$

where $\star$ represents the discrete convolution, and $\tilde{a}_s \in \mathbb{R}^{(s+2\tilde{k}) \times (s+2\tilde{k})}$ represents the discrete signal after zero-padding.

Then the convolution of multi-channel discrete signal $a_s \in \mathbb{R}^{s \times s \times d_{in}}$ and discrete convolution kernel $K_w \in \mathbb{R}^{k \times k \times d_{in} \times d_{out}}$ can be defined as

$$(a_s \star K_w)[i, j, l] = \sum_{m,n=-\tilde{k}}^{\tilde{k}} \sum_{c=1}^{d_{in}} K_w[m, n, c, l] \cdot \tilde{a}_s[i - m, j - n, c], \quad i, j = 1 \ldots s, \tag{A.7}$$

where $l$ corresponds to the index of the output channel and $c$ to the index of the input channel.

**Mamba Integral.** It can be deduced from Eq. (3.10) that

$$F(x) = e^{-Ax} h(x) = \int_{-\infty}^{x} B(e^{-Ay}) v(y) \mathrm{d}y$$

$$= F(0) + \int_0^x B(e^{-Ay}) v(y) \mathrm{d}y \tag{A.8}$$

$$= h(0) + \int_0^x B(e^{-Ay}) v(y) \mathrm{d}y.$$

Continuing the derivation, we can obtain

$$e^{-Ax} h(x) = h(0) + \int_0^x B(e^{-Ay}) v(y) \mathrm{d}y,$$

$$h(x) = e^{Ax} h(0) + e^{Ax} \int_0^x B(e^{-Ay}) v(y) \mathrm{d}y. \tag{A.9}$$

Assume $x_k$ and $x_{k+1}$ are two adjacent sampling points after discretization, we will show below how to go from $h(x_k)$ to $h(x_{k+1})$. Firstly, we provide their definitions below:

$$h(x_k) = e^{Ax_k}h(0) + e^{Ax_k}\int_0^{x_k} B(e^{-Ay})v(y)\mathrm{d}y,$$

$$h(x_{k+1}) = e^{Ax_{k+1}}h(0) + e^{Ax_{k+1}}\int_0^{x_{k+1}} B(e^{-Ay})v(y)\mathrm{d}y,$$

$$= e^{A(x_k+(x_{k+1}-x_k))}h(0) + e^{A(x_k+(x_{k+1}-x_k))}\int_0^{x_{k+1}} B(e^{-Ay})v(y)\mathrm{d}y,$$

$$= e^{A(x_{k+1}-x_k)}[e^{Ax_k}h(0) + e^{Ax_k}\int_0^{x_k} B(e^{-Ay})v(y)\mathrm{d}y] + e^{Ax_{k+1}}\int_{x_k}^{x_{k+1}} B(e^{-Ay})v(y)\mathrm{d}y,$$

$$= e^{A(x_{k+1}-x_k)}h(x_k) + \int_{x_k}^{x_{k+1}} B(e^{A(x_{k+1}-y)})v(y)\mathrm{d}y.$$

(A.10)

Assume $\Delta = x_{k+1} - x_k$ as the step length parameter for the time interval, then

$$h(x_{k+1}) = e^{A\Delta}h(x_k) + \int_{x_k}^{x_{k+1}} B(e^{A(x_{k+1}-y)})v(y)\mathrm{d}y. \qquad (A.11)$$

According to the concept of zero-order hold, when $\Delta$ approaches 0, we can regard the value of function $v$ in the interval $[x_k, x_{k+1}]$ as a constant $v(x_{k+1})$, then Eq. (A.11) can be written as:

$$\begin{aligned}
h(x_{k+1}) &= e^{A\Delta}h(x_k) + \int_{x_k}^{x_{k+1}} e^{A(x_{k+1}-y)}\mathrm{d}y Bv(x_{k+1}), \\
&= e^{A\Delta}h(x_k) + Bv(x_{k+1})e^{Ax_{k+1}}\int_{x_k}^{x_{k+1}} (e^{-Ay})\mathrm{d}y, \\
&= e^{A\Delta}h(x_k) + Bv(x_{k+1})e^{Ax_{k+1}}(-\frac{1}{A}(e^{-Ax_{k+1}} - e^{-Ax_k})), \\
&= e^{A\Delta}h(x_k) + Bv(x_{k+1})\frac{1}{A}(e^{A(x_{k+1}-x_k)} - 1), \\
&= e^{A\Delta}h(x_k) + \Delta Bv(x_{k+1})\frac{1}{\Delta A}(e^{\Delta A} - 1).
\end{aligned} \qquad (A.12)$$

According to the Scharfetter-Gummel method [39], let $\bar{\mathbf{A}} = e^{\Delta A}$, $\bar{\mathbf{B}} = (\Delta A)^{-1}(e^{\Delta A} - \mathbf{I}) \cdot \Delta B$, then:

$$h(x_{k+1}) = \bar{\mathbf{A}}h(x_k) + \bar{\mathbf{B}}v(x_{k+1}). \qquad (A.13)$$

Lastly, with Eq. (3.10) combined, we can obtain:

$$\begin{aligned}
h(x_{k+1}) &= \bar{\mathbf{A}}h(x_k) + \bar{\mathbf{B}}v(x_{k+1}), \\
u(x_{k+1}) &= Ch(x_{k+1}).
\end{aligned} \qquad (A.14)$$

This is easy from Eq. (A.14) to Eq. (3.14).

**Upsampling and Downsampling Operators.** For $w > 0$, let $h_w$ be the interpolation windowed-sinc filter defined as

$$h_w(x_0, x_1) = \mathrm{sinc}(2wx_0) \cdot \mathrm{sinc}(2wx_1), \quad (x_0, x_1) \in \mathbb{R}^2. \qquad (A.15)$$

For a discrete, single-channel signal $a_s \in \mathbb{R}^{s\times s}$, let $(\widetilde{a}_s[n])_{n\in\mathbb{Z}}$ be its periodic extension into infinite length, e.g., $\widetilde{a}_s[n] = a_s[n \bmod s]$ for $n \in \mathbb{Z}$. The discrete upsampling $\mathcal{U}_{s,N} : \mathbb{R}^{s\times s} \to \mathbb{R}^{Ns\times Ns}$ by an integer factor $N \in \mathbb{N}$ of the signal $a_s \in \mathbb{R}^{s\times s}$ is defined by:

1. First, the signal $a_s$ is transformed into $a_{s,\uparrow Ns} \in \mathbb{R}^{Ns\times Ns}$, achieved by interspersing each pair of signal samples from $a_s$ with $N-1$ samples of zero value. The formula is defined as follows:

$$a_{s,\uparrow Ns}[i,j] = \mathbb{I}_S(i) \cdot \mathbb{I}_S(j) \cdot a_s[i \bmod s, j \bmod s], \quad i,j = 1\dots Ns, \qquad (A.16)$$

where $S = \{1, s+1, \dots (N-1)s+1\}$ and $\mathbb{I}_S$ is the indicator function.

2. Second, we use the interpolation filter $h_{s/2}$ to convolve with the output of the first step. The formula is defined as follows:

$$\mathcal{U}_{s,N}(a_s)[i,j] = \sum_{n,m \in \mathbb{Z}} \widetilde{a}_{s,\uparrow Ns}[n,m] \cdot h_{s/2}(is-ns, js-ms), \quad i,j = 1\dots Ns.$$

(A.17)

Please note that the discretization step includes an additional convolution of an interpolation filter, which is more than the up-sampling process of the continuous function. This is because the introduction of zero-value samples expands the signal size while introducing discontinuity, namely high-frequency components. In order to eliminate these high-frequency components, we need to apply interpolation filters to remove these high-frequency noises.

The discrete downsampling $\mathcal{D}_{s,N} : \mathbb{R}^{s \times s} \to \mathbb{R}^{s/N \times s/N}$ by an integer factor $N \in \mathbb{N} \& s/N \in \mathbb{N}$ of the signal $a_s \in \mathbb{R}^{s \times s}$ is defined by:

1. First, we use the interpolation filter $h_{s/2N}$ to convolve with $a_s$. The formula is defined as follows:

$$a_{s,s/N}[i,j] = \sum_{n,m \in \mathbb{Z}} \widetilde{a}_s[n,m] \cdot h_{s/(2N)}(is-ns, js-ms), \quad i,j = 1\dots s/N.$$

(A.18)

2. Second, we resample the discrete signal $a_{s,s/N}$ once every $N-1$ points. The formula is defined as follows:

$$\mathcal{D}_{s,N}(a_s)[i,j] = a_{s,N/s}[(i-1)s+1, (j-1)s+1], \quad i,j = 1\dots s/N.$$

(A.19)

The execution of multi-channel discrete upsampling and downsampling is accomplished by separate applications of the respective single-channel operators.

**Activation layer.** We define the discretized activation function layer as

$$\eta_s(a_s) = \mathcal{D}_{s,N} \circ \sigma \circ \mathcal{U}_{s,N}(a_s),$$

(A.20)

where $\eta : \mathbb{R} \to \mathbb{R}$ is an activation function applied point-wise. Through experimentation, we found that setting $N$ to 2 achieves the best balance between efficiency and effectiveness.

## B  Proofs

### B.1  Proof of Equation 3.12

Start with Eq. (3.10) and differentiate both sides with respect to $x$:

$$\begin{aligned}
h'(x) &= Ae^{Ax} \int_0^x B(e^{-Ay})v(y)\mathrm{d}y + e^{Ax}Be^{-Ax}v(x), \\
&= Ae^{Ax} \int_0^x B(e^{-Ay})v(y)\mathrm{d}y + Bv(x), \\
&= Ah(x) + Bv(x).
\end{aligned}$$

(B.1)

Together with Eq. (3.9), we can obtain Eq. (3.12).

### B.2  Proof that MambaNO is a ReNO.

**Setting.** Follow Eq. (3.4), in MambaNO, one layer is defined as:

$$P_T(\eta_T(W_{T-1} + \mathcal{K}_{T-1} + b_{T-1})).$$

(B.2)

Among them, only $P_T$, $\eta_T$, and $\mathcal{K}_{T-1}$ involve mapping between bandlimited spaces, while $W_{T-1}$ and $b_{T-1}$ are mappings within the bandlimited space. Mappings within the band-limited space do not introduce aliasing errors. Therefore, we only need to discuss the first three operators, e.g., kernel integration operators $\mathcal{K}_{T-1}$, non-linear activation operators $\eta_T$, and projection operators $P_T$. In the following texts, we define $T_{\Psi_w}$ and $T_{\Psi_w}^{\dagger}$ as the synthesis operators and the pseudo-inverses operator between $\mathcal{B}_w$ and $\ell^2(\mathbb{Z}^2)$.

**Kernel Integration Operators.** Firstly, the form of the convolution integration is as follows:

$$\mathcal{K}_w f(x) = \sum_{m,n=-k}^{k} k_{m,n} f(x - z_{m,n}), \quad x \in \mathbb{R}^2, \tag{B.3}$$

where $k \in \mathbb{N}$, $k_{m,n} \in \mathbb{C}$, and $z_{m,n} = \left\{ \left( \frac{m}{2w}, \frac{n}{2w} \right) \right\}_{m,n \in \mathbb{Z}}$. The convolution operator is defined in the form of a finite sum, and each term $f(x - z_{m,n})$ can be regarded as a translation of $f$. Since $f$ is in $f \in \mathcal{B}_w(\mathbb{R}^2)$, these translations are still in $\mathcal{B}_w(\mathbb{R}^2)$. Actions such as multiplying by a finite number of constant coefficients $k_{m,n}$ and taking the sum will not cause the function to escape from this space. From another perspective, if the functions in $\mathcal{B}_w(\mathbb{R}^2)$ have certain properties, such as smoothness and rapid decay, then the weighted sum and movement in convolution will not change the basic properties of the function. Therefore, we present the commutative diagram of the convolution integral:

$$
\begin{array}{ccc}
\mathcal{B}_w & \xrightarrow{\mathcal{K}_w} & \mathcal{B}_w \\
{\scriptstyle T_{\Psi_w}} \uparrow & & \downarrow {\scriptstyle T_{\Psi_w}^\dagger} \\
\ell^2(\mathbb{Z}^2) & \xrightarrow{\sum kf} & \ell^2(\mathbb{Z}^2)
\end{array}
$$

where $\sum kf$ is an abbreviated expression form of $\sum_{m,n=-k}^{k} k_{m,n} f(x - z_{m,n})$.

Secondly, the output of the mamba integration (also known as SSM) can be defined as a global convolution integral like Eq. (3.15). Similar to the defined convolution integral, such a global integral is a well-defined operator from $\mathcal{B}_w$ to itself.

$$
\begin{array}{ccc}
\mathcal{B}_w & \xrightarrow{SSM} & \mathcal{B}_w \\
{\scriptstyle T_{\Psi_w}} \uparrow & & \downarrow {\scriptstyle T_{\Psi_w}^\dagger} \\
\ell^2(\mathbb{Z}^2) & \xrightarrow{\bar{\mathrm{K}}} & \ell^2(\mathbb{Z}^2)
\end{array}
$$

where $\bar{\mathrm{K}}$ is global convolution kernel in Eq. (3.15).

**Non-linear Activation Operators.** Now, we assume the activation function $\eta_{w,\overline{w}}$ as an operator from $\mathcal{B}_w$ to $\mathcal{B}_{\overline{w}}$, given that $\overline{w} > w$. Moreover, $\mathcal{D}_{\overline{w},w} \colon \mathcal{B}_{\overline{w}} \to \mathcal{B}_w$ serves as an orthogonal projection from $\mathcal{B}_{\overline{w}}$ to $\mathcal{B}_w$, while $\mathcal{U}_{w,\overline{w}} \colon \mathcal{B}_w \to \mathcal{B}_{\overline{w}}$ is a natural embedding from $\mathcal{B}_w$ to $\mathcal{B}_{\overline{w}}$. Non-linear activation functions may introduce frequency components beyond the original band-limited space. To reserve spectral space for these potentially introduced high-frequency components, we decide to first boost the frequency of the band-limited space $\mathcal{B}_w$ where the original function $f$ is located, that is $\mathcal{U}_{w,\overline{w}}(f)$. After applying the activation function, in order to maintain system consistency and signal band-limited characteristics, we use $\mathcal{D}_{\overline{w},w}$ to filter out frequency components exceeding $w$, where $\mathcal{D}_{\overline{w},w}$ can be seen as a kind of band-limited filter. Now we can provide the corresponding exchange diagram.

$$
\begin{array}{ccccccc}
\mathcal{B}_w & \xrightarrow{\mathcal{U}_{w,\overline{w}}} & \mathcal{B}_{\overline{w}} & \xrightarrow{\sigma} & \mathcal{B}_{\overline{w}} & \xrightarrow{\mathcal{D}_{\overline{w},w}} & \mathcal{B}_w \\
{\scriptstyle T_{\Psi_w}} \uparrow & & \downarrow {\scriptstyle T_{\Psi_{\overline{w}}}^\dagger} & & {\scriptstyle T_{\Psi_{\overline{w}}}} \uparrow & & \downarrow {\scriptstyle T_{\Psi_w}^\dagger} \\
\ell^2(\mathbb{Z}^2) & \xrightarrow{U_{s,N}} & \ell^2(\mathbb{Z}^2) & \xrightarrow{\sigma} & \ell^2(\mathbb{Z}^2) & \xrightarrow{D_{s,N}} & \ell^2(\mathbb{Z}^2)
\end{array}
$$

In the diagram, $\mathcal{U}_{w,\overline{w}}$ and $\mathcal{D}_{\overline{w},w}$ are discretized as

$$U_{s,N} = T_{\Psi_{\overline{w}}}^* \circ \mathcal{U}_{w,\overline{w}} \circ T_{\Psi_w}, \quad D_{s,N} = T_{\Psi_w}^* \circ \mathcal{D}_{\overline{w},w} \circ T_{\Psi_{\overline{w}}}. \tag{B.4}$$

Therefore, the transformation between the continuous and discrete representations of the activation function $\sigma$ is consistent. This means that the operations we do on the discrete level will not lose information or introduce errors when returning to the continuous level.

**Projection Operators.** As can be seen in Eq. (B.4), the co-existence of up-sampling and down-sampling operators ensures the consistency of band-limited space. Thus, in our implementation, we use the U-shaped architecture, which has an equal number of up-sampling and down-sampling operators. In summary, this exact correspondence between its constituent continuous and discrete operators establishes MambaNO as an example of Representation equivalent neural operators or ReNOs (A.2), thus proving Proposition 3.1 of the main text.

## C Time Complexity Analysis of Other Models

Given a discrete counterpart $u \in R^{H \times W \times C}$ of a two-dimensional continuous function, we can reshape $u$ to get $u \in R^{N \times C}$, with $N = H \times W$.

Table 3: Computational complexity of standard deep learning models. $N$ is the sampling points of a continuous function.

| Models | Range | Complexity |
|--------|-------|------------|
| GNO (kernel) | radius $r$ | $O(N(N-1)))$ |
| Transformer | global | $O(N^2)$ |
| FNO (FFT) | global | $O(NlogN)$ |
| CNO | local | $O(N)$ |
| MambaNO | global+local | $O(N)$ |

**Graph neural operator (GNO).** The integral formula of GNO is:

$$
\begin{aligned}
(\mathcal{K}(v))(x) &= \int_{U(x)} K(x,y) \cdot v(y)\mathrm{d}y, \\
&\approx \sum_{y \in U(x)} K(x,y) \cdot v(y)q_y,
\end{aligned}
\tag{C.1}
$$

where $K$ is a kernel (typically parametrized by a neural network) and $q_y \in \mathbb{R}$ are suitable quadrature weights. While the GNO can represent local integral operators by picking a suitably small neighborhood $U(x) \subset D$, the evaluation of the kernel and aggregation in each neighborhood $U(x)$ is slow and memory-intensive for general kernels $k : D \times D \to \mathbb{R}^n$. For each point, we need to visit each of its adjacent points. As reported in [45], the time complexity of GNO is $O(N(N-1)))$.

**Transformer.** The integral formula of Transformer-based NO is:

$$
(\mathcal{K}(v))(x) = \int_{\Omega} K(v(x), v(y))v(y)\mathrm{d}y,
\tag{C.2a}
$$

$$
(\mathcal{K}(v))(x) \approx \sum_{i=1}^{n_v} K(v(x), v(y_i))v(y_i),
\tag{C.2b}
$$

$$
(\mathcal{K}(v))(x) \approx \sum_{i=1}^{n_v} \frac{\exp\left(\frac{\langle W_q v(x), W_k v(y_i)\rangle}{\sqrt{d_v}}\right)}{\sum_{j=1}^{n_v} \exp\left(\frac{\langle W_q v(x_j), W_k v(y_i)\rangle}{\sqrt{d_v}}\right)} W_v v(y_i).
\tag{C.2c}
$$

from which $K$ could be perceived as employing the softmax function onto three transformed vectors with length $n_v$. Evidently, Eq. (C.2c) agrees closely with the widely adopted self-attention mechanism within transformers, in which the matrices $W_q, W_k, W_v \in \mathbb{R}^{d_v \times d_v}$ respectively represent the learning transformation of queries, keys, and values. As can be seen from Eq. (C.2c), it is necessary to go through two nested loops of length $n_v = N$. Therefore, the time complexity of Transformer-based NO is $O(N^2)$.

**FNO.** FNO replaces the global convolution in the time domain with multiplication operations in the frequency domain, therefore, its kernel integral is:

$$
\begin{aligned}
(\mathcal{K}(v))(x) &= \int_{\Omega} K(x,y)v(y)\mathrm{d}y, \\
&= \int_{\Omega} K(x-y)v(y)\mathrm{d}y, \\
&= \mathcal{F}^{-1}\left(R_\phi \cdot (\mathcal{F}v_t)\right)(x),
\end{aligned}
\tag{C.3}
$$

in which $R_\phi$ designs $N$ multiplications, parametrized by $\phi$, therefore its time complexity is $O(N)$. However, the time complexity of Fast Fourier Transform is $O(NlogN)$, so the total time complexity is $O(NlogN)$.

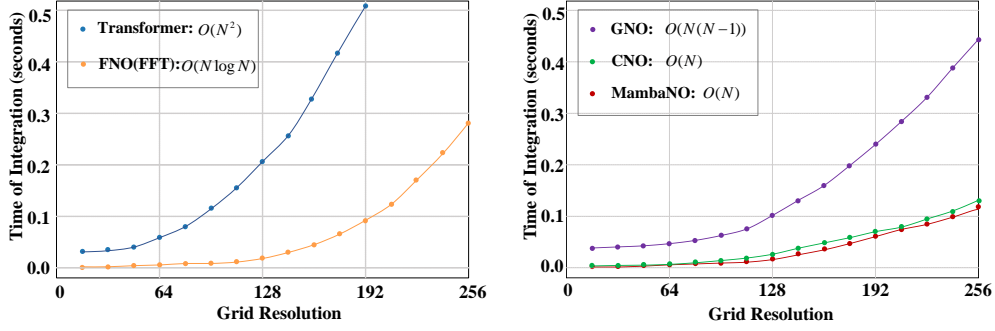

Figure 5: Time of integration vs. Grid resolution.

**CNO.** The local convolution operator is defined by

$$
\begin{aligned}
(\mathcal{K}(v))(x) &= \int_{\Omega} \kappa(x-y)v(y)\mathrm{d}y, \\
&= \sum_{i,j=1}^{k} \kappa_{ij}v(x-z_{ij}),
\end{aligned}
\tag{C.4}
$$

where $f \in \mathcal{B}_w$, $\kappa$ is a discrete kernel with size $k \in \mathbb{N}$, $z_{ij}$ is the resultant grid points. Evaluating each point involves $k^2$ multiplications. So the total time complexity is $O(Nk^2) \Rightarrow O(N)$.

From a quantitative perspective, as shown in Fig. 5, we have measured the time required for different integral forms on different numbers of grids. In which, the specific time complexity of GNO depends on the choice of degree, which is up to $N-1$ in a graph with $N$ points. The graphical representation illustrates that despite the time complexity of $O(N)$ being concurrent for both, the duration necessitated for kernel integration in MambaNO is slightly lower than that required in CNO.

## D  Representative PDE Benchmarks & Numerical Results

**Poisson Equation.** The prototype presented represents a linear elliptic PDE,

$$
-\Delta u = f, \text{ in } D, \quad u|_{\partial D} = 0.
\tag{D.1}
$$

The solution operator $\mathcal{G}^{\dagger}$, mapping the source term $f$ to the solution $u$, can be denoted as $\mathcal{G}^{\dagger} : f \mapsto u$. Let us consider the source term:

$$
f(x,y) = \frac{\pi}{K^2} \sum_{i,j=1}^{K} a_{ij} \cdot (i^2+j^2)^{-r} \sin(\pi i x) \sin(\pi j y), \quad \forall (x,y) \in D,
\tag{D.2}
$$

with $r = -0.5$. While training the models, we set $K = 16$ in Eq. (D.2) and select $a_{ij}$ to possess a uniform distribution between $[-1, 1]$. Beyond in-distribution testing, we also conduct out-of-distribution testing by configuring $K = 20$.

For the training phase, we produce 1024 samples, and for the evaluation phase, we furnish 256 samples each for both in-distribution and out-of-distribution testing. These are attained by sampling the precise solution, $u$, at a resolution of $64 \times 64$ points on $D = [0,1]^2$. A validation set, composed of 128 samples, is also constructed for model selection purpose. The data utilized for training is normalized to the range $[0,1]$ with the same normalization constants as the training data. In Fig. 6, a random in-distribution testing sample and an out-of-distribution testing sample, as well as predictions made by FNO, CNO, and MambaNO, are depicted. When handling the Poisson equation, FNO is obviously inferior to CNO and MambaNO. In terms of handling details, MambaNO is superior to CNO.

**Wave Equation.** The given prototype represents a linear hyperbolic PDE,

$$
u_{tt} - c^2 \Delta u = 0, \text{ in } D \times (0, T), \quad u_0(x,y) = f(x,y),
\tag{D.3}
$$

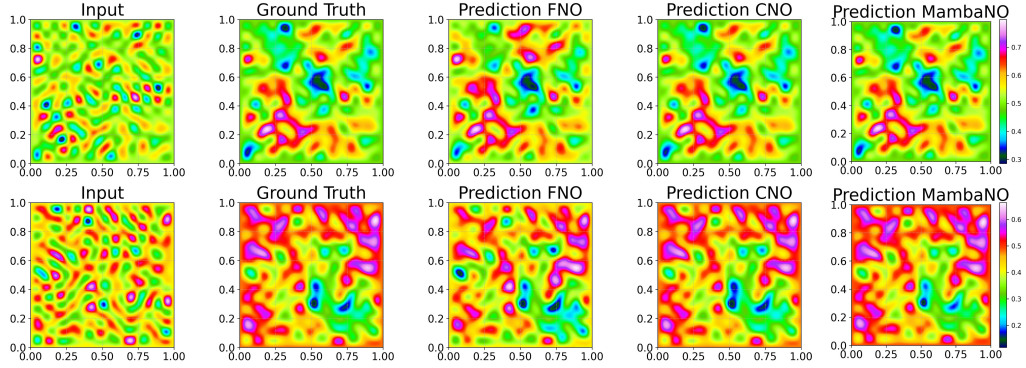

Figure 6: Visual predictions on representative in- (top row) and out-of-distribution (bottom row). (Poisson Equation.)

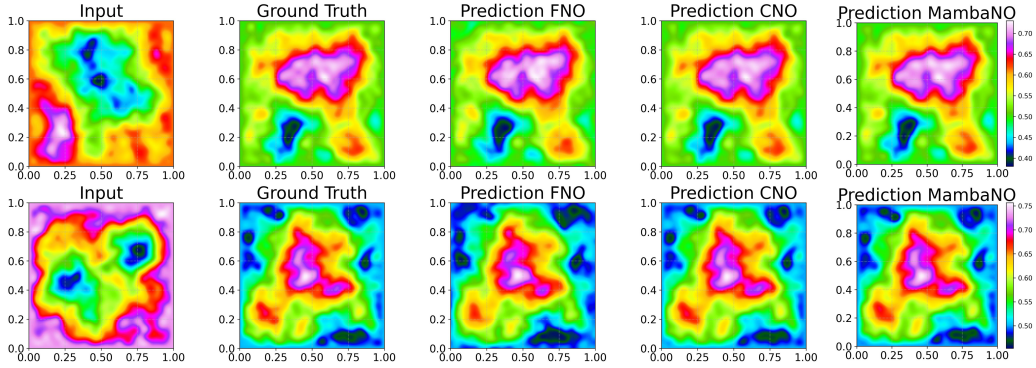

Figure 7: Visual predictions on representative in- (top row) and out-of-distribution (bottom row). (Wave Equation.)

with a constant propagation speed $c = 0.1$. This represents a multiscale standing wave exhibiting periodic pulsations over time, which are dependent on the variable $K$. Considering the initial conditions defined by Eq. (D.2) where $r = 1$ and $K = 24$, it becomes possible to accurately compute the exact solution. The exact value at time $t > 0$ is given by

$$u(x,y,t) = \frac{\pi}{K^2} \sum_{i,j}^{K} a_{ij} \cdot (i^2 + j^2)^{-r} \sin(\pi ix) \sin(\pi jy) \cos\left(c\pi t \sqrt{i^2 + j^2}\right), \quad \forall (x,y) \in D.$$

(D.4)

Our goal is to approximate the function at the moment when $T = 5$. For generating training and in-distribution test instances, we configure $T = 5$, $K = 24$, and assign $a_{ij}$ a uniform distribution value from $[-1, 1]$. Conversely, for out-of-distribution evaluation, we adjust the decay exponent of the modes to $r = 0.85$ and $K = 32$.

A total of 512 samples are generated for the training set, and 256 samples each are generated for in-distribution and out-of-distribution testing by sampling the precise solution at a $64 \times 64$ resolution. We generate 128 samples to create a separate validation set. All training data are normalized to fit within the $[0, 1]$ interval, and the testing data are normalized using the same constants as those employed for the training data. In Fig. 7, a random in-distribution testing sample and an out-of-distribution testing sample, as well as predictions made by FNO, CNO, and MambaNO, are given. The predictive performance of CNO is better than FNO, but our MambaNO is even better.

**Transport Equation.** The PDE is represented as follows,

$$u_t + v \cdot \nabla u = 0, \quad u(t = 0) = f,$$

(D.5)

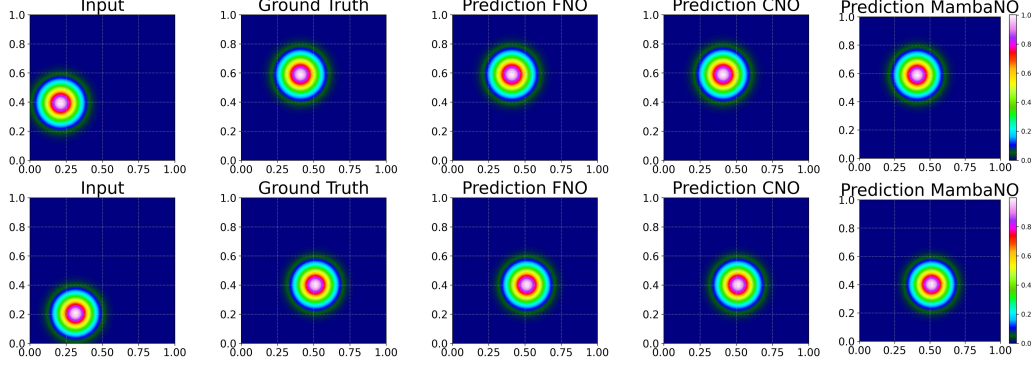

Figure 8: Visual predictions on representative in- (top row) and out-of-distribution (bottom row). (Smooth Transport.)

with a given velocity field and initial data $f$. By defining the constant velocity field $v$ as $v = (v_x, v_y) = (0.2, 0.2)$, the solution $u(x, y, t)$ is obtained as $u(x, y, t) = f(x - v_x t, y - v_y t)$. We consider two distinctive training data types, i.e., Smooth Transport and Discontinuous Transport. Furthermore, in both instances, we generate 512 samples for training, 256 for validation, and 256 for both in-distribution and out-of-distribution testing, all derived from the precise solution. Every sample is normalized to fit within the interval $[0, 1]$.

**Smooth Transport.** The first kind comprises smooth initial data, configured as a radially symmetric Gaussian. Its centers are chosen randomly and uniformly from the range $(0.2, 0.4)^2$, and the corresponding variance is consistently drawn from the set $(0.003, 0.009)$. Formally, the initial conditions are given by

$$f(\underline{x}) = \frac{1}{\sqrt{(2\pi)^2 \det(\Sigma)}} \exp\left(-\frac{1}{2}(\underline{x} - \mu)^T \Sigma^{-1}(\underline{x} - \mu)\right), \quad \underline{x} = (x, y), \quad \mu = (\mu_x, \mu_y), \quad \text{(D.6)}$$

where $\Sigma = \sigma I$ such that $\sigma \sim \mathcal{U}(0.003, 0.009)$ and $\mu_x, \mu_y \sim \mathcal{U}(0.2, 0.4)$. $I$ is an identity matrix and $\mathcal{U}(\cdot)$ is the uniform distribution. Every sample is normalized to fit within the interval $[0, 1]$. For out-of-distribution testing, the centers of the Gaussian inputs are uniformly sampled from the range $(0.4, 0.6)^2$, implying that $\mu_x$ and $\mu_y$ are distributed uniformly between 0.4 and 0.6. The data is generated at $64 \times 64$ resolution. In Fig. 8, a random in-distribution testing sample and an out-of-distribution testing sample, as well as predictions made by FNO, CNO, and MambaNO, are provided. The figures further validate the conclusions that MambaNO outperforms CNO and FNO.

**Discontinuous Transport.** The second variety is a discontinuous initial data, designed as the indicator function of a radial disk. Here, the centers are uniformly pulled from $(0.2, 0.4)^2$, and the radii are chosen uniformly from the span $(0.1, 0.2)$. For out-of-distribution testing, the disk centers are pulled uniformly from $(0.4, 0.6)^2$. Formally, the initial conditions are given by

$$f(\underline{x}) = \mathbb{I}_{S_r(\mu)}(\underline{x}), \quad \underline{x} = (x, y), \quad \mu = (\mu_x, \mu_y), \quad \text{(D.7)}$$

where $r \sim \mathcal{U}(0.1, 0.2)$, $\mu_x, \mu_y \sim \mathcal{U}(0.2, 0.4)$, $\mathbb{I}$ is an indicator function, and $S_r(\mu) = \{\underline{x} : \|\underline{x} - \mu\|_2 \leq r\}$ is the sphere of radius $r$ with the center $\mu$. Due to the infinite spectral content in discontinuous data, the aliasing error tends to persist throughout data sampling. To alleviate this, firstly, we generate samples at a $128 \times 128$ resolution. Then, by performing a frequency domain downsampling of the generated data, we obtain the actual samples at a resolution of $64 \times 64$. In Fig. 9, a random in-distribution testing sample and an out-of-distribution testing sample, as well as predictions made by FNO, CNO, and MambaNO, are depicted. The accompanying figures substantiate the conclusions in Table 1, concurrently demonstrating a consistent superiority of MambaNO over both CNO and FNO in testing scenarios.

**Allen-Cahn Equation.** This serves as a prototype for nonlinear parabolic PDEs,

$$u_t = \Delta u - \varepsilon^2 u(u^2 - 1), \quad \text{(D.8)}$$

with a reaction rate of $\varepsilon = 220$. This refers to the underlying operator $\mathcal{G}^\dagger : f \mapsto u(., T)$ which maps the initial conditions denoted by $f$ to the final solution $u$ recorded at a final time $T$ equal to 0.0002.

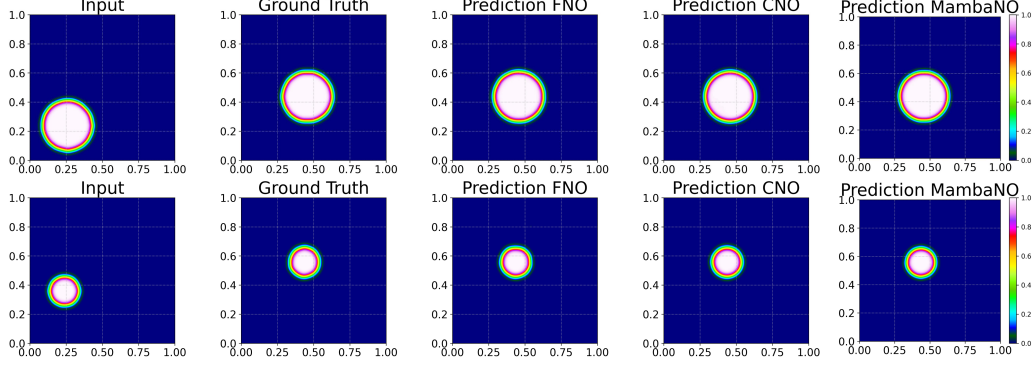

Figure 9: Visual predictions on representative in- (top row) and out-of-distribution (bottom row). (Discontinuous Transport.)

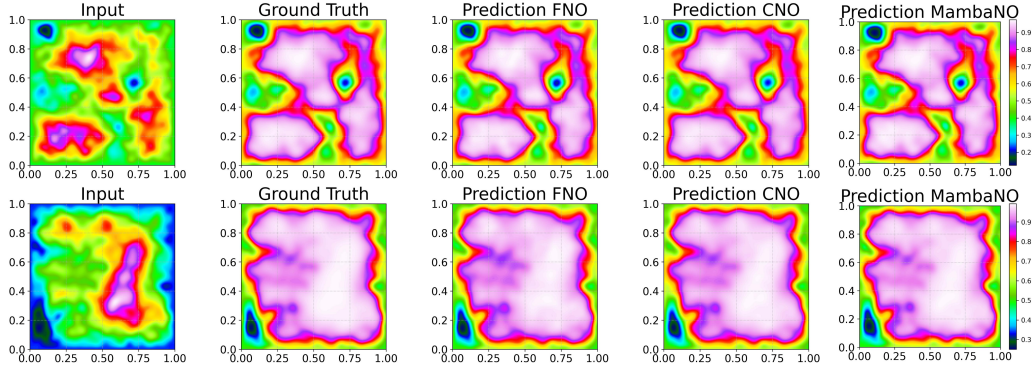

Figure 10: Visual predictions on representative in- (top row) and out-of-distribution (bottom row). (Allen-Cahn Equation.)

The prescribed initial conditions for both training and in-distribution testing conform to the format given in Eq. (D.2), where $r$ equals 1, $K$ equals 24, and coefficients $a_{ij}$ are uniformly drawn from the range $[-1, 1]$. For out-of-distribution testing, $K$ is set to 16, and the initial decay, denoted by $r$, is randomly selected from a uniform distribution within the range $[0.85, 1.15]$ of the modes in Eq. (D.2). This setup facilitates the evaluation of the model's capacity to infer varying dynamics of the system. Training and testing data are generated using a finite difference scheme on a grid, with a resolution of $64^2$. Specifically, space is uniformly discretized at resolution $s^2 = 64 \times 64$, with $\Delta x$ defined as $1/s$. Subsequently, given that an explicit method is employed, the time domain is uniformly discretized with a time step $\Delta t$ approximately equal to $5.47 \times 10^{-7}$. $N$ is set equal to $\lfloor T/\Delta t \rfloor + 1$. Here, $U_{i,j}^n$ signifies $u(i\Delta x, j\Delta x, n\Delta t)$ for $i, j = 0, 1, ..., s$ and $n = 0, 1, ..., N$. Additionally, zero-valued ghost cells are incorporated at the boundaries. The finite difference scheme is delineated as follows:

$$U_{i,j}^{n+1} = U_{i,j}^n + \frac{\Delta t}{\Delta x} \left( U_{i+1,j}^n + U_{i,j+1}^n + U_{i-1,j}^n U_{i,j-1}^n - 4U_{i,j}^n \right) - \Delta t \varepsilon^2 U_{i,j}^n \left( U_{i,j}^n \cdot U_{i,j}^n - 1 \right),$$
(D.9)

for $i, j = 0, 1, ..., s$ and $n = 0, 1, ..., N$. The selected $\Delta t$ meets the CFL condition, ensuring $\Delta t < \frac{(\Delta x)^2}{2\varepsilon}$. The generation process yields 256 training samples, 128 validation samples, and 128 each of in-distribution and out-of-distribution testing samples, maintaining a resolution of $64 \times 64$. Normalization of the training data is performed within the range $[0, 1]$. The testing data are normalized using the same constants as those used for the training data. In Fig. 10, a random in-distribution testing sample and an out-of-distribution testing sample, as well as predictions made by FNO, CNO, and MambaNO, are provided. Reaffirming the conclusions drawn from Table 1, these figures again demonstrate that MambaNO leads a slight edge compared to CNO and FNO.

**Navier-Stokes Equation.** The referenced PDEs provide a model for the motion of incompressible fluids, with the formative principles being

$$u_t + (u \cdot \nabla)u + \nabla p = \nu \Delta u, \quad \text{div } u = 0, \tag{D.10}$$

for the torus $D = \mathbb{T}^2$, subject to periodic boundary conditions and a viscosity of $\nu = 4 \times 10^{-4}$, which are applied solely to Fourier modes of an amplitude $\geq 12$. This approach effectively models fluid flow under conditions of an extremely high Reynolds number. This refers to the underlying operator $\mathcal{G}^\dagger : f \mapsto u(., T)$ which maps the initial conditions denoted by $f$ to the final solution $u$ recorded at a final time $T$ equal to 1. Initially, we adopt the following conditions.

$$u_0(x, y) = \begin{cases} \tanh\left(2\pi \frac{y - 0.25}{\rho}\right) & \text{for } y + \sigma_\delta(x) \leq \frac{1}{2} \\ \tanh\left(2\pi \frac{0.75 - y}{\rho}\right) & \text{otherwise} \end{cases} \tag{D.11}$$

$$v_0(x, y) = 0$$

where $\sigma_\delta : [0, 1] \to \mathbb{R}$ is a variable of the initial data provided by

$$\sigma_\delta(x) = \delta \sum_{k=1}^{p} \alpha_k \sin(2\pi k x - \beta_k). \tag{D.12}$$

The random variables $\alpha_k$ and $\beta_k$ are independently and identically distributed, with uniform distribution on the intervals $[0, 1]$ and $[0, 2\pi]$ respectively. The parameters $\delta$ and $p$ are set as $\delta = 0.025$ and $p = 10$. For the smoothing parameter, we select $\rho = 0.1$. Our initial conditions reference the widely recognized thin shear layer issue. Here, the evolution of the shear layer occurs via vortex shedding, leading to a complex array of vortices. The creation of training and in-distribution testing samples employs a spectral viscosity method, initiating with a sinusoidal perturbation of the shear layer. The layer exhibits a thickness of $\rho = 0.1$ and comprises 10 perturbation modes, each uniformly sampled from the interval $[-1, 1]$. For the experiments conducted outside of the distribution, we reduce $\rho$ to $\rho = 0.09$ and relocate the shear layers closer to the midpoint of the domain, positioning them at $y = 0.3$ and $y = 0.7$, as opposed to the locations $y = 0.25$ and $y = 0.75$ in the original initial condition. This examination assesses the model's capacity to accommodate a flow regimen with a higher count, and variable locations, of the shed vortices. For the training set, we produce a total of 750 samples and generate an additional 128 samples for use in the validation set, as well as in-distribution and out-of-distribution testing. The Navier-Stokes equations are simulated with a spectral viscosity method on a $128 \times 128$ resolution to generate both the training and testing data, and then these data are downsampled to a $64 \times 64$ resolution. The target is to discern the operator that bridges the initial velocity to velocity at $T = 1$. The training data are normalized within the range $[0, 1]$, whilst the testing data employ the training data's normalization constants for its own normalization. In Fig. 3, a random in-distribution testing sample and an out-of-distribution test sample are given, as well as predictions made by FNO, CNO, and MambaNO. From the figure, we can see that MambaNO obviously performs better than FNO and CNO, particularly in the parts within the black box.

**Darcy flow.** The steady-state Darcy flow is described by the second order linear elliptic PDE,

$$-\nabla \cdot (a\nabla u) = f, \text{ in } D, \quad u|_{\partial D} = 0. \tag{D.13}$$

In this equation, $a$ represents the diffusion coefficient, while $f = 1$ denotes the forcing term. The solution operator $\mathcal{G}^\dagger$, symbolized as $a \mapsto u$, establishes a mapping from the diffusion coefficient $a$ (expressed as a push forward of a Gaussian process) to the solution $u$. In this context, the input is the diffusion coefficient $a$, denoted as $a \sim \psi\#\mu$. Here, $\mu$ signifies a Gaussian Process with a zero mean and a squared exponential kernel

$$k(x, y) = \sigma^2 \exp\left(\frac{|x - y|^2}{l^2}\right), \quad \sigma^2 = 0.1. \tag{D.14}$$

The function $\psi : \mathbb{R} \to \mathbb{R}$ assigns a value of 12 to positive components on the real line and 3 to the negative elements. For this particular scenario, the push-forward measure is characterized by a pointwise definition. The in-distribution and out-of-distribution samples are characterized by variations in the length scales of the Gaussian process. For in-distribution testing, a length scale of $l = 0.1$ is utilized. While for out-of-distribution tests, we employ a length scale of $l = 0.05$. For

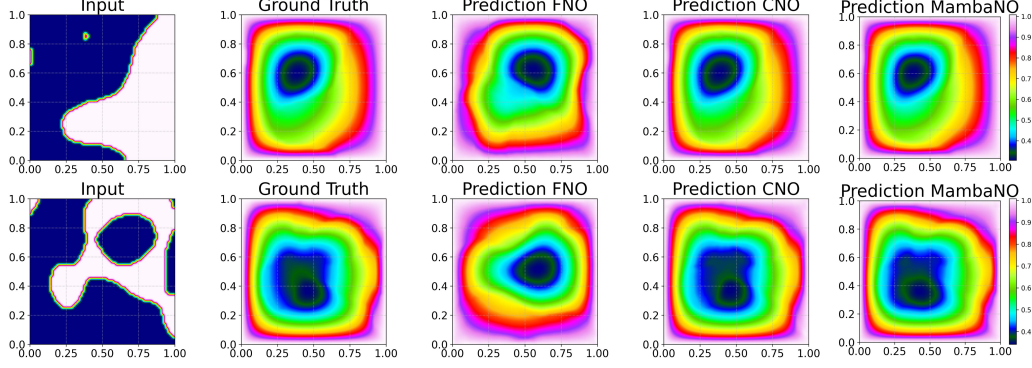

Figure 11: Visual predictions on representative in- (top row) and out-of-distribution (bottom row). (Darcy flow.)

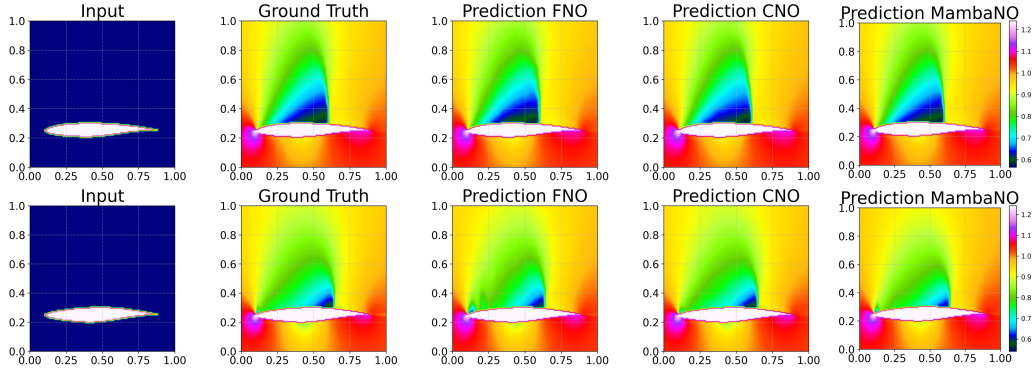

Figure 12: Visual predictions on representative in- (top row) and out-of-distribution (bottom row). (Flow past airfoils.)

the training session, 256 samples are generated. Additionally, the study sees the generation of 128 samples for a validation set, in-distribution, and out-of-distribution testing. The data corresponds to a resolution of $64 \times 64$. In Fig. 11, a random in-distribution testing sample and an out-of-distribution testing sample, as well as predictions made by FNO, CNO, and MambaNO, are listed. The graphical representation clearly demonstrates that the performance of FNO cannot catch up those of CNO or MambaNO. In finer details, our model exhibits superior performance compared to CNO, thus corroborating the values presented in Table 1.

**Flow past airfoils.** This flow is represented utilizing the compressible Euler equations in our model,

$$u_t + \text{div} F(u) = 0, u = [\rho, \rho v, E]^{\perp}, F = [\rho v, \rho v \otimes v + p\mathbf{I}, (E + p)]v]^{\perp}. \tag{D.15}$$

The density $\rho$, velocity $v$, pressure $p$, and total energy $E$ are interrelated in the modular framework, guided by an ideal gas equation of state,

$$E = \frac{1}{2}\rho|u|^2 + \frac{p}{\gamma - 1}, \tag{D.16}$$

where $\gamma = 1.4$. Other significant variables related to flow complicate matters, such as the speed of sound $a = \sqrt{\frac{\gamma p}{\rho}}$ and the Mach number $M = \frac{|u|}{a}$. The airfoils under consideration are characterized by perturbing to the established shape of the RAE2822 airfoil, implemented using Hicks-Henne Bump functions. Specifically, in alignment with standard protocols in aerodynamic shape optimization, we examine a reference airfoil shape where the upper and lower surface of the airfoil are situated at $(x, y_{\text{ref}}^{\text{U}}(x/c))$ and $(x, y_{\text{ref}}^{\text{L}}(x/c))$ respectively. Here, $c$ represents the chord length while $y_{\text{ref}}^{\text{U}}$ and $y_{\text{ref}}^{\text{L}}$ correspond to the renowned RAE2822 airfoil. The reference shape subsequently undergoes

perturbations through the application of the Hicks-Henne Bump functions,

$$y^{\mathrm{L}}(\xi) = y_{\mathrm{ref}}^{\mathrm{L}}(\xi) + \sum_{i=1}^{10} a_i^{\mathrm{L}} B_i(\xi), \quad y^{\mathrm{U}}(\xi) = y_{\mathrm{ref}}^{\mathrm{U}}(\xi) + \sum_{i=1}^{10} a_i^{\mathrm{U}} B_i(\xi),$$

which satisfies

$$\begin{cases} B_i(\xi) = \sin^3(\pi \xi^{q_i}), \\ \quad q_i = \dfrac{\ln 2}{\ln 14 - \ln i}, \\ \quad \xi = \dfrac{x}{c}, \\ a_i^{\mathrm{L}} = 2(\psi_i - 0.5)(i+1) \times 10^{-3}, \\ a_i^{\mathrm{U}} = 2(\psi_{i+10} - 0.5)(11-i) \times 10^{-3}, \\ \quad i = 1, ..., 10, \\ \quad \psi \in [0,1]^d. \end{cases}$$

We can now formally categorize the airfoil shape as $\mathcal{S} = \{(x,y) \in D : x \in [0,c], y^L \leq y \leq y^U\}$ and consequently, designate the shape function $f = \chi_{[\mathcal{S}]}(x,y)$, with $\chi$ symbolizing the characteristic function. The foundational operator in focus, $\mathcal{G}^{\dagger} : f \mapsto \rho$, transitions the shape function $f$ into the flow's density at the steady state of compressible Euler equations. The freestream boundary conditions

$$T^{\infty} = 1, \quad M^{\infty} = 0.729, \quad p^{\infty} = 1, \quad \alpha = 2.31°$$

are employed, with the solution operator projecting the shape function onto the persistent state density distribution. The data is ultimately interpolated onto a Cartesian grid of dimensions $128 \times 128$ on the underlying domain $D = [-0.75, 1.75]^2$. Unit values are subsequently allocated to the density $\rho(x,y)$ for all coordinates $(x,y)$ within the set $S$. The forms of the training data samples align with 20 bump functions, with coefficients $\psi$ uniformly drawn from $[0,1]^2$. Testing beyond the range of distribution is conducted with 30 bump functions. Throughout the training and assessment stages, the discrepancy between the acquired solution and the actual reality is solely computed for the points $(x,y)$ that exist outside the airfoil shape $S$. We produce 750 samples for the training set and 128 samples each for the validation set, in-distribution testing set, and out-of-distribution testing set. In this controlled examination, the data are not standardized. In Fig. 12, a random in-distribution testing sample and an out-of-distribution testing sample, as well as predictions made by FNO, CNO, and MambaNO, are depicted. Whether in general or in local details, MambaNO has achieved the best performance.

## E    More Ablation Studies

Please note, all the experimental data are relative to our default choice, which is marked with $*$. From Fig. 1, it can be seen that MambaNO assumes a function input and throws it into a set of encoders, where the input is space-wisely downsampled but channel-wisely widened. The depth of the encoder determines how many scale-varied feature functions are within the model. Regarding the *Depth*, we design the ablation experiments concerned, with the results shown in Table 4. When *Depth* $= 4$, the model achieves the best balance among performance, inference time, parameters, and FLOPs.

As mentioned in the main body of the text, we expand the band-limited space of the original function using an upsampling operation before employing the activation operator. Therefore, the ablation experiments investigating the optimal value of upsampling factor $N_f$ are also conducted. The results are given in Table 4. As can be seen, the case with no upsampling, i.e., $N_f$=1, suffers severe aliasing errors. while a fourfold upsampling provides a minor improvement, it significantly increases the computational burden. Therefore, we consider that a twofold upsampling rate achieves a well balance of efficacy and efficiency.

Meanwhile, in our cross-scan module, we used four different directional scans for the function. We wanted to understand the specific impact of different directional scans on model performance enhancement, so we conducted related ablation experiments, the results are as shown in the Table 5.We use the variable $direction$ to represent the number of cross-scan directions in the experiment. Evidently, we observe a superior effect when the $direction = 4$, thus motivating our selection of this value.

Finally, we conducted relevant ablation experiments on the number of integrations in the encoder and decoder, as shown in the Table 6. Here, $[x, y, m, n]$ represents the number of integrations in the encoder or decoder in Fig. 1, from top to bottom. When we discuss the encoder, the decoder is fixed to the default selection with $*$; similarly, when discussing the decoder, the encoder is the default selection. Our model is the best choice among all configurations.

Table 4: The ablation results by using different parameters. * indicates our default choice.

| Parameter | Test errors ↓ | Time ↓ | Params ↓ | FLOPs ↓ |
|---|---|---|---|---|
| a. *Depth* = 2 | 6.07× | 0.63× | 0.08× | 0.61× |
| b. *Depth* = 3 | 2.94× | 0.81× | 0.28× | 0.82× |
| c. *Depth* = 4* | 1.00× | 1.00× | 1.00× | 1.00× |
| d. *Depth* = 5 | 1.30× | 1.21× | 3.73× | 1.17× |
| a. $N_f$=1 | 1.50× | 0.78× | 0.87× | 0.83× |
| b. $N_f$=2* | 1.00× | 1.00× | 1.00× | 1.00× |
| c. $N_f$=4 | 0.97× | 2.33× | 1.52× | 1.33× |

Table 5: The ablation results by using different amounts of directions in Cross-Scan. * indicates our default choice

| Cross-Scan Directions | Test errors↓ | Time↓ | Params↓ | FLOPs↓ |
|---|---|---|---|---|
| a. *direction*=1 | 1.51× | 0.74× | 0.81× | 0.75× |
| b. *direction*=2 | 1.22× | 0.85× | 0.87× | 0.84× |
| c. *direction*=4* | 1.00× | 1.00× | 1.00× | 1.00× |

Table 6: The ablation results by using different amounts of integration. * indicates our default choice.

| Encoder's Integration Layers | | Test errors↓ | Time↓ | Params↓ | FLOPs↓ |
|---|---|---|---|---|---|
| Conv Integration | Mamba Integration | | | | |
| a. [1,1,1,1] | a. [1,1,1,1] | 1.91× | 0.77× | 0.65× | 0.67× |
| b. [2,2,2,2]* | b. [2,2,2,2]* | 1.00× | 1.00× | 1.00× | 1.00× |
| c. [3,3,3,3] | c. [3,3,3,3] | 1.05× | 1.94× | 1.35× | 1.33× |

| Decoder's Mamba Integration Layers | Test errors↓ | Time↓ | Params↓ | FLOPs↓ |
|---|---|---|---|---|
| a. [1,1,1,1] | 1.58× | 0.71× | 0.86× | 0.85× |
| b. [2,2,2,2]* | 1.00× | 1.00× | 1.00× | 1.00× |
| c. [3,3,3,3] | 1.20× | 1.53× | 1.14× | 1.15× |

# F   More Comparative Experiments

In Table 4, the occasional performance decline observed with increasing model capacity can be attributed to underfitting due to insufficient data samples. To address this, we conducted comparative experiments on a larger dataset[33] (10,000 samples, with 7,000 used for training), evaluating the same four O(N) complexity algorithms: Oformer [11], GNOT [12], CNO [13], and our MambaNO. The quantitative results are presented in Table 7, and we also provide efficiency metrics for a more in-depth comparison. As shown in Fig. 13, the accuracy of our model has further improved. Overall, MambaNO still leads in performance among the four O(N) competitors, while ranking second in terms of epochs used and training time, and third in terms of GPU memory usage (close to CNO).

# G   Hyperparameters

For all competing methods, we used the optimal parameters recommended in the original papers. As for the proposed MambaNO, most of the hyperparameters, including encoder depth, upsampling factor, scanning direction, and integration layers, have been ablated, with results and practical settings

Table 7: (Left) Relative median $L^1$ test errors for OFormer, GNOT, CNO and MambaNO on the 2DCFD dataset (The compressible Navier-Stokes equations in 'PDEBENCH: An Extensive Benchmark for Scientific Machine Learning'). (Right) MambaNO's performance on the Navier-Stokes equations under different dimension hyperparameters.

| | In/Out | OFormer | GNOT | CNO | MambaNO | Dimensions | | 8 | 16 | 32 | 48 |
|---|---|---|---|---|---|---|---|---|---|---|---|
| 2DCFD | In | 1.71% | 1.42% | 1.35% | **1.29%** | Navier-Stokes | In | 4.34% | 2.74% | 2.70% | 3.21% |
| | Out | 3.32% | 3.22% | 2.57% | **2.13%** | Equation | Out | 8.52% | 5.95% | 5.92% | 7.14% |
| GPU memory usage | | 15736M | 5718M | 8433M | 8858M | GPU memory usage | | 5694M | 8854M | 15536M | 22364M |
| Epochs | | 329 | 817 | 999 | 407 | Epochs | | 999 | 802 | 851 | 907 |
| Total Time | | 4.57h | 5.22h | 4.93h | 4.75h | Total Time | | 3.11h | 3.27h | 6.18h | 15.02h |

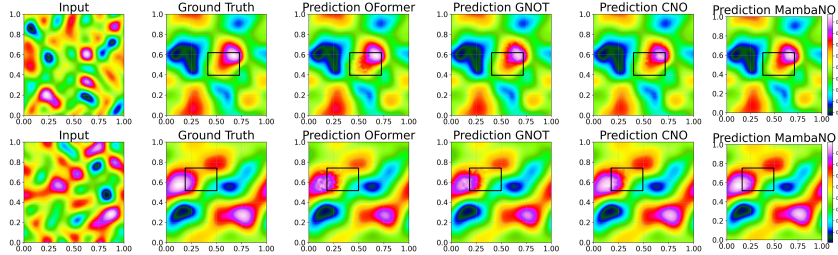

Figure 13: Visual predictions on representative in- (top row) and out-of-distribution (bottom row).

Table 8: MambaNO best-performing hyperparameters configuration for different benchmarks. $\eta$, $\gamma$, and $\omega$ represent the learning rate, scheduler gamma, and weight decay, respectively. Convolution Integration and Mamba Integration denote the quantities of each in a single layer of kernel integration.

| | Batch Size | Epochs | $\eta$ | $\gamma$ | $\omega$ | Dimensions | Convolution Integration | Mamba Integration | Params (M) | FLOPs (G) |
|---|---|---|---|---|---|---|---|---|---|---|
| Poisson Equation | 16 | 1000 | 0.001 | 0.98 | 1e-06 | 16 | 2 | 1 | 1.10 | 0.30 |
| Wave Equation | 16 | 1000 | 0.001 | 0.98 | 1e-10 | 32 | 2 | 1 | 4.33 | 1.01 |
| Smooth Transport | 16 | 1000 | 0.001 | 0.98 | 1e-06 | 16 | 2 | 2 | 1.38 | 0.36 |
| Discontinuous Transport | 16 | 1000 | 0.001 | 0.98 | 1e-06 | 16 | 2 | 2 | 1.38 | 0.36 |
| Allen-Cahn | 16 | 1000 | 0.001 | 0.98 | 1e-06 | 32 | 2 | 2 | 5.40 | 1.23 |
| Navier-Stokes | 16 | 1000 | 0.001 | 0.98 | 1e-10 | 16 | 2 | 2 | 1.38 | 0.36 |
| Darcy Flow | 16 | 1000 | 0.001 | 0.98 | 1e-06 | 32 | 2 | 2 | 5.40 | 1.23 |
| Compressible Euler | 16 | 1000 | 0.001 | 0.98 | 1e-10 | 32 | 3 | 2 | 6.97 | 1.53 |

provided in subsection E of the supplemental material. The remaining training parameters are listed in Table 8. In terms of different PDEs, most parameters were fixed across all experiments, with some fine-tuned for better performance.

# H Broader Impacts

Our work will improve the speed and accuracy of solving PDE, which is beneficial to promote the progress of related disciplines such as mathematics and physics; it has no negative impact on society.

